# Gated Inference Network: Inference and Learning State-Space Models

**Hamidreza Hashempoor**
Seoul National University
Department of Electrical and Computer Engineering
`hamidreza.hashemp@snu.ac.kr`

**Wan Choi**
Seoul National University
Department of Electrical and Computer Engineering
`wanchoi@snu.ac.kr`

## Abstract

This paper advances temporal reasoning within dynamically changing high-dimensional noisy observations, focusing on a latent space that characterizes the nonlinear dynamics of objects in their environment. We introduce the *Gated Inference Network* (GIN), an efficient approximate Bayesian inference algorithm for state space models (SSMs) with nonlinear state transitions and emissions. GIN disentangles two latent representations: one representing the object derived from a nonlinear mapping model, and another representing the latent state describing its dynamics. This disentanglement enables direct state estimation and missing data imputation as the world evolves. To infer the latent state, we utilize a deep extended Kalman filter (EKF) approach that integrates a novel compact RNN structure to compute both the Kalman Gain (KG) and smoothing gain (SG), completing the data flow. This design results in a computational cost per step that is linearly faster than EKF but introduces issues such as the exploding gradient problem. To mitigate the exploding gradients caused by the compact RNN structure in our model, we propose a specialized learning method that ensures stable training and inference. The model is then trained end-to-end on videos depicting a diverse range of simulated and real-world physical systems, and outperforms its counterparts —RNNs, autoregressive models, and variational approaches— in state estimation and missing data imputation tasks.

## 1 Introduction

A state-space model is a type of graphical model that effectively represents noise-afflicted data [1]. Typically, the objective is to conduct inference, which involves obtaining accurate estimates of the posterior distribution of latent states or noise-free measurements. In videos, inferring the state space presents a complex challenge owing to the high dimensionality of noisy observations, which only offer partial information about the states. Previous methods have proposed various inference approaches such as sampling [2], variational inference [3], or belief propagation [4]. Within the framework of a hidden Markov process, classical methods like the celebrated Kalman filter (KF) and smoother [5, 6], along with their extensions like EKF and UKF, [7, 8] have been employed to address the posterior inference problem. However, the reliance of these methods on accurate knowledge of the underlying dynamics, coupled with computationally intensive matrix inversions, poses challenges in scalability to high-dimensional latent spaces.

To address these issues, this paper distinguishes between the sensory observation and its dynamics by disentangling two representations: a transformed observation $\mathbf{w}_t$ obtained by mapping the original sensory observation $\mathbf{o}_t$ through a nonlinear function modeled by neural networks, and the latent state space $\mathbf{x}_t$ describing the dynamics of $\mathbf{w}_t$ at time step $t$. To infer high dimensional state space (dynamics) $\mathbf{x}_t$, we introduce filtering and smoothing techniques that build upon classical EKF updates but incorporate flexible function estimators without relying on a constrained graphical model. We model temporal dynamics with a linearized Gaussian SSM, which is adapted to accommodate complex dynamics. To address non-linearity and the multiplicity of dynamics, we learn multiple transition and emission matrices, each modeling a unique dynamical scenario. Then, the weight of each matrix during inference is determined by the past state $\mathbf{x}_{t-1}$ through a neural network. To prevent mode collapse and ensure the system's ability to capture diverse dynamics, we introduce a loss term proportional to the KL divergence of the transition matrices during training. This design enables our model to handle various state scenarios, similar to approaches found in the switching linear dynamics systems (SLDS) literature [9]. Unlike EKF, which requires precise characterization and modeling of the underlying dynamics, we operate under the assumption that noise statistics and the underlying physical dynamics of SSM are entirely unknown.

The main drawback of the EKF is that it takes $\mathcal{O}(n^3)$ time per step, because we need to invert the posterior covariance matrix with size of $n$. This makes the method slow to use. Moreover, we identify the computation of the KG and SG in the (extended) KF as critical components that rely on noise statistics and domain knowledge. In this paper, we efficiently model KG and SG using GRU cells, bypassing the need for matrix inversion in the KF flow. Consequently, the GIN algorithm achieves a time complexity of $\mathcal{O}(\beta^2 n^2)$ per step, making it linearly faster than EKF. To capitalize on the sparsity of the covariance matrix, we employ a convolutional approach that further reduces the size of $n$ relative to $\beta$, where $0 < \beta < 1$. However, the nonlinear structure of GRU cells can lead to gradient explosion [10], a phenomenon that occurs when the dynamics undergo significant changes as GRU parameters traverse bifurcation points [11] during the learning process. To address this issue, we propose a learning method based on an analysis of GRU cell dynamics. This method aims to prevent parameter values from reaching bifurcation points by stabilizing the dynamics of GRU cells, thereby addressing the issue of gradient explosion.

To summarize, our primary contribution is a novel algorithm designed for the recursive inference of SSMs with arbitrary nonlinear (and possibly non-Gaussian) emission and transition models, with high-dimensional noisy stimuli observation. This algorithm utilizes a linearized Gaussian dynamics approach, with parameter estimation conducted through neural networks. This means that we can apply the recursive Bayesian updates, akin to the Kalman filter and Kalman smoother. We introduce a likelihood for inferring the states (dynamics) and another likelihood for inferring the high-dimensional images (of video). The objective is maximized in a supervised manner, depending on the task. We propose learning schemes for GRU cells to address issues related to gradient explosion and instability. Finally, we introduce a loss term proportional to the KL divergence of the learned transition matrices to prevent the system from becoming stuck in mode collapse.

To substantiate our claims, we conduct five experiments. First, we simulate a nonlinear dynamic system using the pendulum sequence video, a common benchmark in this literature, to demonstrate our model's ability to infer both dynamics and images. Second, we introduce a more challenging experiment of a simulated nonlinear double pendulum, where the video sequence is heavily distorted with noise, showcasing our model's resilience of dynamics estimation and image imputation to noisy observations. Third, we present a switching dynamics irregular bouncing ball experiment to illustrate our model's ability to handle multiple dynamic scenarios. Fourth, we perform visual odometry using the KITTI dataset, demonstrating the practical applicability of our methods in real-world scenarios. Finally, we assess the effectiveness of our proposed gradient explosion handling scheme by evaluating convergence across various simulation seeds.

## 2 Related Works

EKF and UKF represent early extensions of the original Kalman filter, allowing for nonlinear transitions and emissions. These methods, categorized as model-based (MB) algorithms, rely on precise knowledge and modeling of the underlying dynamics within a fully characterized SSM. Consequently, the performance of these MB methods is heavily contingent on the accuracy of domain knowledge and model assumptions. Recent approaches, such as BackpropKF [12] and SIN [13], perform EKF in the latent state using deep encoders for feature extraction. However, they face similar

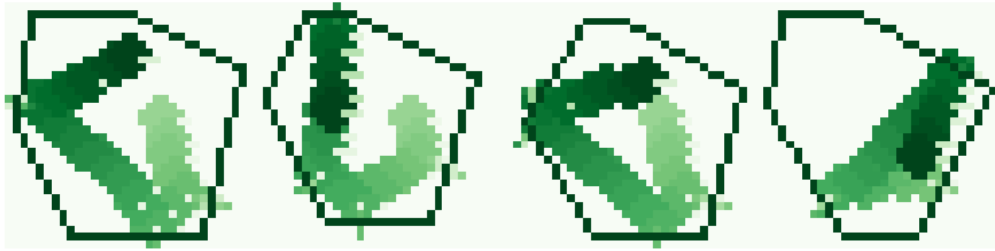

Figure 1: Generated sequences from GIN in the irregular polygon environments. The videos are shown as single images, with color intensity representing the incremental sequence index.

limitations as EKF, including computational complexity and scalability. To address these issues, a common approach involves considering a diagonal covariance matrix in the KF flow as in [14] and [15]. However, we aim to find a richer approximation to the posterior.

A significant body of work, including Variational Auto-Encoders (VAE) [3], Embed to Control (E2C) [16], and Importance Weighted VAE (IWVAE) [17], integrates deep learning with variational inference (VI), but lacks memory cells and recurrent structures for handling imputation tasks. To address this, EM-based VI approaches like Structure VAE (SVAE) [18], Kalman VAE (KVAE) [19], Disentangled VAE (DVAE) [20], Extended KVAE (EKVAE) [21], Robust VAE [22], and Markovian VAE (MVAE) [23] use original KF equations for filtering and smoothing. However, they cannot directly optimize states (dynamics), as noted in RKN [24] and CRU [25]. Classical memory networks like LSTMs [26], GRUs [27], and simple RNNs [28] infer latent states but fail to provide insights into uncertainties and dynamics.

LatentNet [29], KalmanNet [30] and SSI-SSM [31] utilize GRU in their structures for state updates, similar to the parameterization used in the GIN. However, these models typically necessitate access to complete or partial dynamics information, i.e. they are MB, and their reliance on vanilla GRU cells can cause instability of the SSM. Liu et al. [32] proposes a disentanglement model for audio data, which shares similarities with the GIN. However, their parameterization and objectives differ from those of the GIN, and they do not utilize compact RNNs in their inference structure.

We cover the most relevant works in this section. A more detailed discussion, including a comprehensive comparison of recent related works like System Identification (SI) with details of EM algorithm, auto-regressive (AR) and SLDS models, is provided in Appendix A.8.3 and Table 7. This builds upon the findings of the RKN with additional complements. We also conduct an empirical complexity analysis in appendix A.8.2, to evaluate the computational efficiency of our method compared to the discussed approaches. This analysis involved measuring the execution time per iteration using a clock on the wall as the benchmark.

## 3 Background

**Linear Gaussian state space models.** Linear Gaussian state space models (LGSSMs) are commonly employed to model vectors of observed time series, denoted as $\mathbf{w} = [\mathbf{w}_1, ..., \mathbf{w}_T]$. LGSSMs excel in filtering and smoothing tasks. They model the first-order Markov process on the state space $\mathbf{x} = [\mathbf{x}_1, ..., \mathbf{x}_T]$, which may also incorporate external control inputs $\mathbf{u} = [\mathbf{u}_1, ..., \mathbf{u}_T]$ under the multivariate normality assumption of the state:

$$p_{\gamma_t}(\mathbf{x}_t|\mathbf{x}_{t-1}, \mathbf{u}_t) = \mathcal{N}(\mathbf{x}_t; \mathbf{F}_t\mathbf{x}_{t-1} + \mathbf{B}_t\mathbf{u}_t, \mathbf{Q}_t), \quad p_{\gamma_t}(\mathbf{w}_t|\mathbf{x}_t) = \mathcal{N}(\mathbf{w}_t; \mathbf{H}_t\mathbf{x}_t, \mathbf{R}_t). \quad (1)$$

$\gamma_t$ represents the parameters of the system at time $t$, encapsulating information from matrices $\mathbf{F}_t, \mathbf{B}_t, \mathbf{H}_t, \mathbf{Q}_t$ and $\mathbf{R}_t$, which correspond to the state transition, control, emission, process noise covariance, and observation noise covariance, respectively. At each time step, the transition and emission procedures are subject to distortion caused by process noise and observation noise, following distributions of $\mathcal{N}(\mathbf{0}, \mathbf{Q}_t)$ and $\mathcal{N}(\mathbf{0}, \mathbf{R}_t)$, respectively.

**Filtering and smoothing algorithms.** The goal of filtering is to infer the posterior distribution $p_{\gamma_{1:t}}(\mathbf{x}_t|\mathbf{w}_{1:t}, \mathbf{u}_{1:t})$, while, the smoothing aims to infer the smoothing distribution $p_{\gamma_{1:T}}(\mathbf{x}_t|\mathbf{w}_{1:T}, \mathbf{u}_{1:T})$ given the whole observations. Without loss of generality, we drop the input variable $\mathbf{u}$ in the rest of paper, while one can condition all of the distributions on the input variable in the case of its existence. Considering the prior state parameterization $p_{\gamma_{1:t}}(\mathbf{x}_t|\mathbf{w}_{1:t-1}) = \mathcal{N}(\mu_{t|t-1}, \mathbf{\Sigma}_{t|t-1})$ and relying on [33], the filtering and smoothing parameterizations are as:

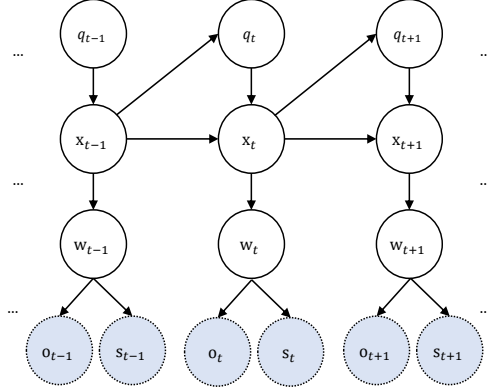

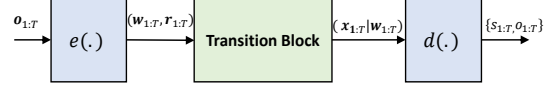

Figure 3: The GIN as a HW model for system identification. By appropriate structure selection for $e(.)$ and $d(.)$, the GIN can handle high dimensional observations. The relation between the internal variables, $\mathbf{w}_t$ and $\mathbf{x}_t$, is simulated by the transition block.

Figure 2: Graphical model. Dashed nodes are task dependent output.

$$p_{\gamma_{1:t}}(\mathbf{x}_t|\mathbf{w}_{1:t}) = \mathcal{N}\left(\mu_{t|t-1} + \mathbf{K}_t[\mathbf{w}_t - \mathbf{H}_t\mu_{t|t-1}], \ \ \mathbf{\Sigma}_{t|t-1} - \mathbf{K}_t\mathbf{S}_f\mathbf{K}_t^T\right) = \mathcal{N}\left(\mu_{t|t}, \mathbf{\Sigma}_{t|t}\right) \quad (2)$$

$$p_{\gamma_{1:T}}(\mathbf{x}_t|\mathbf{w}_{1:T}) = \mathcal{N}\left(\mu_{t|t} + \mathbf{J}_t[\mu_{t+1|T} - \mathbf{F}_{t+1}\mu_{t|t}], \ \ \mathbf{\Sigma}_{t|t} + \mathbf{J}_t\mathbf{S}_s\mathbf{J}_t^T\right) = \mathcal{N}(\mu_{t|T}, \mathbf{\Sigma}_{t|T}) \quad (3)$$

$$\mathbf{K}_t = \mathbf{\Sigma}_{t|t-1}\mathbf{H}_t^T.\left[\mathbf{H}_t\mathbf{\Sigma}_{t|t-1}\mathbf{H}_t^T + \mathbf{R}_t\right]^{-1}, \qquad \mathbf{J}_t = \mathbf{\Sigma}_{t|t}\mathbf{F}_{t+1}^T\mathbf{\Sigma}_{t+1|t}^{-1} \quad (4)$$

with $\mathbf{S}_f = \left[\mathbf{H}_t\mathbf{\Sigma}_{t|t-1}\mathbf{H}_t^T + \mathbf{R}_t\right]$ and $\mathbf{S}_s = \left[\mathbf{\Sigma}_{t+1|T} - \mathbf{\Sigma}_{t+1|t}\right]$. In (4), $\mathbf{K}_t$ and $\mathbf{J}_t$ are KG and SG, respectively. See appendix A.1 for a full derivation of filtering-smoothing distributions.

## 4   Gated Inference Network

The background section defines key elements for the GIN: original observations $\mathbf{o}_{1:T}$, task-dependent output which can be either the original denoised-imputed observations $\mathbf{o}_{1:T}$ or physical system's states $\mathbf{s}_{1:T}$, transferred observations $\mathbf{w}_{1:T} \in \mathbb{R}^{m \times T}$, latent states (dynamics) $\mathbf{x}_{1:T} \in \mathbb{R}^{n \times T}$, noise covariance of transferred observations $\mathbf{R}_{1:T}$ with diagonals $\mathbf{r}_{1:T} \in \mathbb{R}^{m \times T}$, noise covariance of states process $\mathbf{Q}_{1:T}$ with diagonals $\mathbf{q}_{1:T} \in \mathbb{R}^{n \times T}$, and the parameters of the SSM $\gamma_{1:T}$. We consider the dimensions of the transferred observation and state at each time step to be $m$ and $n$, respectively. The states evolve through transition distribution $p_{\gamma_{1:T}}(\mathbf{x}_{1:T}) = p(\mathbf{x}_1)\prod_{t=2}^T p_{\gamma_t}(\mathbf{x}_t|\mathbf{x}_{t-1})$. Each $\mathbf{w}_t$ is assumed to be drawn from noisy emission probability $p_{\gamma_t}(\mathbf{w}_t|\mathbf{x}_t)$, then the generative process is assumed to factorize as $p_{\gamma_{1:T}}(\mathbf{x}_{1:T}, \mathbf{w}_{1:T}) = p_{\gamma_{1:T}}(\mathbf{x}_{1:T})\prod_{t=1}^T p_{\gamma_t}(\mathbf{w}_t|\mathbf{x}_t)$. This factorisation imposes emission conditional independence like $\mathbf{w}_t \perp\!\!\!\perp (\mathbf{w}_{-t}, \mathbf{x}_{-t})|\mathbf{x}_t$, where $(\mathbf{w}_{-t}, \mathbf{x}_{-t}) = (\mathbf{w}_{(1:t-1,t+1:T)}, \mathbf{x}_{(1:t-1,t+1:T)})$. Common models that assume conditional independence include linear dynamical systems, hidden Markov models, but also nonlinear SSMs with higher-order Markov chains in latent space. Graphical model of the GIN is in figure 2, where output (one of colored nodes) is generated, contingent upon the task. In all of our scenarios, the parameters of the SSM, $\gamma_{1:T}$, are completely unknown.

From SI perspective, the GIN operates similarly to a Hammerstein-Wiener (HW) model [34], employing non-linear transfer functions $e(.)$ and $d(.)$ (Figure 3). Leveraging encoder-decoder structures for $e(.)$ and $d(.)$, the GIN conducts the transfer between original sensory observations $\mathbf{o}_{1:T}$ and lower-dimensional representations $\mathbf{w}_{1:T}$ and task dependent output. The transition block in Figure 3 evolves states using the proposed deep EKF approach, efficiently approximating filtering-smoothing distributions while ensuring system stability with the imposed constraint. Further details and formulations are provided in the inference section. The transition block depicted in Figure 4.

## 5   Parameterization and Inference

Given original observations $\mathbf{o}_{1:T}$ and transferred observations $\mathbf{w}_{1:T}$, our aim is to infer the latent states $\mathbf{x}_{1:T}$. To accomplish this, we seek to infer the marginal distributions

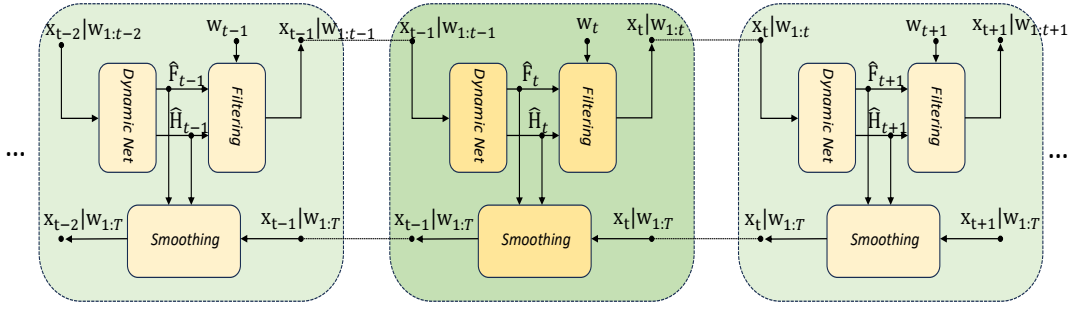

Figure 4: Transition block of figure 3 in details. In each time step, the last posterior $\mathbf{x}_{t-1}|\mathbf{w}_{1:t-1}$ is fed to the *Dynamic Net* to compute $\gamma_t$. In the filtering steps, by using the last posterior $\mathbf{x}_{t-1}|\mathbf{w}_{1:t-1}$ and the observation $\mathbf{w}_t$, the next posterior $\mathbf{x}_t|\mathbf{w}_{1:t}$ is obtained. Having posterior $\mathbf{x}_t|\mathbf{w}_{1:t}$ and the next smoothing state $\mathbf{x}_{t+1}|\mathbf{w}_{1:T}$, applying smoothing for the current state is feasible so that the smoothing state $\mathbf{x}_t|\mathbf{w}_{1:T}$ is obtained.

$p_{\gamma_{1:t}}(\mathbf{x}_t|\mathbf{w}_{1:t})$ for the online inference approach (filtering) and $p_{\gamma_{1:T}}(\mathbf{x}_t|\mathbf{w}_{1:T})$ for the full inference approach (smoothing). We introduce an advantageous prediction parameterization as $p_{\gamma_t}(\mathbf{x}_t|\mathbf{x}_{t-1},\mathbf{w}_{1:t-1}) = \mathcal{N}(\mathbf{F}_t\mathbf{x}_{t-1},\mathbf{Q}_t)$, where $\mathbf{x}_{t-1}$ is sampled from the last filtering distribution, $p_{\gamma_{1:t-1}}(\mathbf{x}_{t-1}|\mathbf{w}_{1:t-1})$. Then, we obtain the prior distribution at time $t$, $p_{\gamma_{1:t}}(\mathbf{x}_t|\mathbf{w}_{1:t-1}) = \mathcal{N}(\mathbf{F}_t\mu_{t-1|t-1},\mathbf{F}_t\Sigma_{t-1|t-1}\mathbf{F}_t^T + \mathbf{Q}_t) = \mathcal{N}(\mu_{t|t-1},\Sigma_{t|t-1})$, by marginalizing out $\mathbf{x}_{t-1}$ from $\int p_{\gamma_t}(\mathbf{x}_t|\mathbf{x}_{t-1},\mathbf{w}_{1:t-1})p_{\gamma_{1:t-1}}(\mathbf{x}_{t-1}|\mathbf{w}_{1:t-1})d\mathbf{x}_{t-1}$ integration. The Gaussianity of $p_{\gamma_{1:t}}(\mathbf{x}_t|\mathbf{w}_{1:t-1})$ results from the Gaussianity of prediction parameterization. After obtaining $p_{\gamma_{1:t}}(\mathbf{x}_t|\mathbf{w}_{1:t-1})$ and observing $\mathbf{w}_t$, it is feasible to derive the filtering parameterization using equation (2). Once all transferred observations $\mathbf{w}_{1:T}$ are available, backward induction can be employed to propagate to previous states using the chain rule. This procedure, known as smoothing, is parameterized with (3). These parameterizations provide valuable insights into two key aspects: 1) appropriately modeling $\gamma$ using neural networks, and 2) illustrating a tractable method to parameterize KG and SG and construct distributions approximations. These approximations serve the basis for constructing output.

**Learning $\gamma$.** To handle multiple dynamic scenarios, we learn $K$ sets of state transition and emission matrices $\check{\mathbf{F}}^k$ and $\check{\mathbf{H}}^k$, each representing a distinct dynamic status. These matrices are combined with state-dependent coefficients $\alpha^k(\mathbf{x}_{t-1})$ as :

$$\hat{\mathbf{F}}_t = \sum_{k=1}^{K} \alpha^k(\mathbf{x}_{t-1})\check{\mathbf{F}}^k, \quad \hat{\mathbf{H}}_t = \sum_{k=1}^{K} \alpha^k(\mathbf{x}_{t-1})\check{\mathbf{H}}^k. \tag{5}$$

We model $\alpha^k(\mathbf{x}_{t-1})$ with a $K$ dimension softmax output neural network called the *Dynamic Net*. It takes the last filtering state $\mathbf{x}_{t-1}$ as input, containing the system's history with lower noise distortion than the transferred observations $\mathbf{w}_{t-1}$. This choice enhances noise robustness, demonstrated in our experiments with time-correlated noise (See Appendix A.6).

In the graphical model shown in Figure 2, we observe two paths for belief propagation from $\mathbf{x}_{t-1}$ to $\mathbf{x}_t$. The first path, $\mathbf{x}_{t-1} \rightarrow \mathbf{x}_t$, linked with $\hat{\mathbf{F}}_t$. The second path involves an intermediate variable $\mathbf{q}_t \sim \mathcal{N}(\mathbf{0},\mathbf{Q}_t)$: $\mathbf{x}_{t-1} \rightarrow \mathbf{q}_t \rightarrow \mathbf{x}_t$. Since learned $\hat{\mathbf{F}}_t$ transfers information from $\mathbf{x}_{t-1}$ to $\mathbf{x}_t$, we argue that it can capture effects similar to those of $\mathbf{Q}_t$, as both are intended to convey message between $\mathbf{x}_{t-1}$ and $\mathbf{x}_t$ [1]. By incorporating $\hat{\mathbf{F}}_t$ from equation (5) into the prior state distribution described earlier as $p_{\gamma_{1:t}}(\mathbf{x}_t|\mathbf{w}_{1:t-1}) = \mathcal{N}(\mathbf{F}_t\mu_{t-1|t-1},\mathbf{F}_t\Sigma_{t-1|t-1}\mathbf{F}_t^T + \mathbf{Q}_t)$, we can neglect the $\mathbf{Q}_t$ term in the covariance, as its influence is accounted for by the learned $\hat{\mathbf{F}}_t$. There are two other meaningful parameterizations for the process noise matrix. First involves direct parameterization by $\mathbf{x}_{t-1}$ using a neural network. Second approach is to parameterize it recursively using $\mathbf{x}_{t-1}$ and $\mathbf{q}_{t-1}$, resulting in a new graphical model with an edge from $\mathbf{q}_{t-1} \rightarrow \mathbf{q}_t$ (See Appendix A.2). Both parameterizations are included in our results, presented in the appendix, to show their effectiveness.

Finally, the diagonal elements of the transferred observation noise vector $\mathbf{r}_t$ are directly mapped from the original observation space using the encoder function $e(.)$ shown in Figure 3. The mapping employs an activation function, elu+1, to handle the positivity of the diagonal elements.

**Filtering and Smoothing Approximation.** We consider $GRU^{KG}$ network to approximate KG as:

$$\hat{\mathbf{K}}_t = \hat{\Sigma}_{t|t-1}\hat{\mathbf{H}}_t^T\mathbf{M}_t\mathbf{M}_t^T, \quad \mathbf{M}_t = GRU^{KG}\big([\text{Conv}(\hat{\Sigma}_{t|t-1}),\mathbf{r}_t]\big) \tag{6}$$

$$\hat{\mu}_{t|t} = \hat{\mu}_{t|t-1} + \hat{\mathbf{K}}_t.[\mathbf{w}_t - \hat{\mathbf{H}}_t\hat{\mu}_{t|t-1}], \qquad \hat{\mathbf{\Sigma}}_{t|t} = \hat{\mathbf{\Sigma}}_{t|t-1} + \hat{\mathbf{K}}_t.[\hat{\mathbf{H}}_t.\hat{\mathbf{\Sigma}}_{t|t-1}.\hat{\mathbf{H}}_T^T + \mathbf{R}_t].\hat{\mathbf{K}}_t^T \qquad (7)$$

In Equation (6), $\mathrm{Conv}(.)$ represents a zero bias convolutional layer with pooling, employed to deal with sparsity of covariance matrix and extract its relevant information while reducing its size. The $[,]$ symbol denotes the concatenation operator. Furthermore, the presence of a positive $\mathbf{r}_t$ vector and the consideration of the Cholesky factor, $\mathbf{M}_t\mathbf{M}_t^T$ in (6), ensure the resulting covariance matrices maintain positive definiteness. This parameterization construct a new filtering distribution $q_{\gamma_{1:t}}(\mathbf{x}_t|\mathbf{w}_{1:t}) = \mathcal{N}(\hat{\mu}_{t|t}, \hat{\mathbf{\Sigma}}_{t|t})$ that is an approximation of (2). Consequently, we consider the approximated prior distribution as $q_{\gamma_{1:t}}(\mathbf{x}_t|\mathbf{w}_{1:t-1}) = \mathcal{N}(\hat{\mathbf{F}}_t\hat{\mu}_{t-1|t-1}, \hat{\mathbf{F}}_t\hat{\mathbf{\Sigma}}_{t-1|t-1}\hat{\mathbf{F}}_t^T) = \mathcal{N}(\hat{\mu}_{t|t-1}, \hat{\mathbf{\Sigma}}_{t|t-1})$, where $(\hat{\mu}_{t|t-1}, \hat{\mathbf{\Sigma}}_{t|t-1})$ are used in (7).

After obtaining filtering states from $q_{\gamma_{1:t}}(\mathbf{x}_t|\mathbf{w}_{1:t})$, we use $GRU^{SG}$ network to approximate SG in a similar way we used $GRU^{KG}$ in (6) as:

$$\hat{\mathbf{J}}_t = \hat{\mathbf{\Sigma}}_{t|t}\hat{\mathbf{F}}_{t+1}^T\mathbf{N}_t\mathbf{N}_t^T, \qquad \mathbf{N}_t = GRU^{SG}\big(\mathrm{Conv}(\hat{\mathbf{\Sigma}}_{t+1|t})\big) \qquad (8)$$

$$\hat{\mu}_{t|T} = \hat{\mu}_{t|t} + \hat{\mathbf{J}}_t\big[\hat{\mu}_{t+1|T} - \hat{\mathbf{F}}_{t+1}\hat{\mu}_{t|t}\big], \qquad \hat{\mathbf{\Sigma}}_{t|T} = \hat{\mathbf{\Sigma}}_{t|t} + \hat{\mathbf{J}}_t\big(\hat{\mathbf{\Sigma}}_{t+1|T} - \hat{\mathbf{F}}_{t+1}\hat{\mathbf{\Sigma}}_{t|t}\hat{\mathbf{F}}_{t+1}^T\big)\hat{\mathbf{J}}_t^T \qquad (9)$$

where the first smoothing state is set to the last filtering state. The new smoothing distribution $q_{\gamma_{1:T}}(\mathbf{x}_t|\mathbf{w}_{1:T}) = \mathcal{N}(\hat{\mu}_{t|T}, \hat{\mathbf{\Sigma}}_{t|T})$ is an approximation of the exact smoothing distribution in (3). For a GRU cell with an input size of $i$ and a hidden state size of $h$, the computational complexity in the forward pass is $\mathcal{O}(3h(h + i + 3))$, which scales linearly with the input size [35]. Considering $\mathrm{Conv}(\mathbf{\Sigma}) \in \mathbb{R}^{\beta^2 n^2}$ as the input of our GRU cells in (6) and (8) with $\beta \in [0, 1]$ as the pooling ratio, the forward pass of GIN for one time step has a time complexity of $\mathcal{O}(3h\beta^2 n^2)$, where $n \gg h$. Compared with LGSSM matrix inversion time complexity of $\mathcal{O}(n^3)$, GIN is faster by a factor of $\frac{n}{3h\beta^2}$, which is crucial in high-dimensional regimes.

# 6 Fitting

In the state estimation task, the output from $d(.)$ in figure 3 equals $\mathbf{s}$, the physical system's states. However, in the imputation task, output of $d(.)$ is same as $\mathbf{o}$ since the original observation is reconstructed (see decoder structure for each task in A.8.1). The conditional distributions $p(\mathbf{w}_{1:T}|\mathbf{o}_{1:T})$ and $q_{\gamma_{1:T}}(\mathbf{x}_{1:T}|\mathbf{w}_{1:T})$ are modeled using an encoder $e(.)$ and smoothing parameterization, respectively. Meanwhile, the conditional distributions $p(\mathbf{s}_{1:T}|\mathbf{x}_{1:T}, \mathbf{w}_{1:T}, \mathbf{o}_{1:T})$ and $p(\mathbf{o}_{1:T}|\mathbf{x}_{1:T}, \mathbf{w}_{1:T})$ are represented by the decoder $d(.)$ for the tasks of state estimation and imputation. These distributions are modeled using multivariate Gaussian and Bernoulli distributions, respectively. Depending on the characteristics of the observations and states, alternative likelihood distributions can also be employed. For instance, the beta likelihood for data in the unit interval, mixtures for multiple marginal distributions, and the negative-binomial likelihood for positive count data (See appendix A.3).

**Likelihood for Inferring States.** The following theorem defines objective for inferring physical system's states. The proof is provided in appendix A.4.1.

**Theorem 1.** *The lower bounded conditional log likelihood of the physical system's states given original images is determined as:*

$$\mathcal{L}(\mathbf{s}_{1:T}|\mathbf{o}_{1:T}) = \frac{1}{N}\sum_{i=1}^{N} log\, p(\mathbf{s}_{1:T}|\mathbf{x}_{1:T}^{(i)}, \mathbf{w}_{1:T}^{(i)}, \mathbf{o}_{1:T}) = \frac{1}{N}\sum_{i=1}^{N}\sum_{t=1}^{T} log\, \mathcal{N}\left(\mathbf{s}_t \Big| d_m(\hat{\mu}_{t|T}^{(i)}), d_c(\hat{\mathbf{\Sigma}}_{t|T}^{(i)})\right) \qquad (10)$$

*where the $d_m(.)$ and $d_c(.)$ determines those parts of the decoder obtaining the state mean and variance. $N$ sequences of $(\mathbf{x}_{1:T}^{(i)}, \mathbf{w}_{1:T}^{(i)}) \sim q_{\gamma_{1:T}}(\mathbf{x}_{1:T}, \mathbf{w}_{1:T}|\mathbf{o}_{1:T})$ are sampled for Monte Carlo integration estimation.*

**Likelihood for Inferring Images.** For the imputation task, consider the ground truth as the sequence of images $\mathbf{o}_{1:T}$ with the dimension of $D_o$. The following theorem defines objective for inferring images. See appendix A.4.2 for the proof.

**Theorem 2.** *The lower bound of log likelihood of the original images is:*

$$\mathcal{L}(\mathbf{o}_{1:T}) = \frac{1}{N}\sum_{i=1}^{N} log\, p(\mathbf{o}_{1:T}|\mathbf{x}_{1:T}^{(i)},\mathbf{w}_{1:T}^{(i)}) = \frac{1}{N}\sum_{i=1}^{N}\sum_{t=1}^{T}\sum_{k=1}^{D_o}\mathbf{o}_t^{(k)}log\big(d_k(\hat{\mu}_{t|T}^{(i)})\big)+\big(1-\mathbf{o}_t^{(k)}\big)log(1-d_k(\hat{\mu}_{t|T}^{(i)}))$$

(11)

*with $N$ sequences of $(\mathbf{x}_{1:T}^{(i)},\mathbf{w}_{1:T}^{(i)}) \sim q_{\gamma_{1:T}}(\mathbf{x}_{1:T},\mathbf{w}_{1:T}|\mathbf{o}_{1:T})$ for Monte Carlo integration estimation. $d_k(.)$ defines the corresponding part of the decoder that maps the $k$-th pixel of $\mathbf{o}_t$ image.*

We use Wishart distribution as a prior for our covariance matrix of states in (10) and (11), which pushes the covariance toward a scale of identity matrix. Such prior prevents getting high log-likelihood due to the high uncertainty. We shrink this scale toward zero as time passes, as we expect the model to finally perform with very little uncertainty, approaching deterministically.

**Mode collapse handling.** To address the issue of mode collapse, where the model becomes stuck in the same state, we propose a loss term on $(i,j)$ pair of transition matrices as follows:

$$l_{\mathrm{mc}}(i,j) = ||\check{\mathbf{F}}^i - \mathrm{Diag}(\mathbf{m}_i)||_F^2 - ||\check{\mathbf{F}}^i - \check{\mathbf{F}}^j||_F^2 - ||\check{\mathbf{F}}^i - \mathrm{Diag}(\mathbf{m}_j)||_F^2 \tag{12}$$

with $\mathbf{m}_i$ and $\mathbf{m}_j$ being distinct hyper parameter vectors, and $||.||_F$ is the Frobenius norm. The term $-\frac{1}{2}\sum_{i=1}^{K}\sum_{j=1,j\neq i}^{K} l_{\mathrm{mc}}(i,j)$ can be added into (10) and (11). This addition ensures that each $\check{\mathbf{F}}^i$ captures a unique transition, distinct from others. From a statistical perspective, (12) represents a term proportional to $-KL\big(p_i(\check{\mathbf{F}})||p_j(\check{\mathbf{F}}).pri_j(\check{\mathbf{F}})\big) + KL\big(p_i(\check{\mathbf{F}})||pri_i(\check{\mathbf{F}})\big)$ (See appendix A.5). $p_i(\check{\mathbf{F}}) \sim \mathcal{MN}_{n\times n}(\check{\mathbf{F}}^i,\mathbf{I},\mathbf{I})$ and $p_j(\check{\mathbf{F}}) \sim \mathcal{MN}_{n\times n}(\check{\mathbf{F}}^j,\mathbf{I},\mathbf{I})$, are matrix normal distributions with priors $pri_i(\check{\mathbf{F}}) \sim \mathcal{MN}_{n\times n}(\mathrm{Diag}(\mathbf{m}_i),\mathbf{I},\mathbf{I})$ and $pri_j(\check{\mathbf{F}}) \sim \mathcal{MN}_{n\times n}(\mathrm{Diag}(\mathbf{m}_j),\mathbf{I},\mathbf{I})$, respectively.

**Gradient explosion handling.** Training (10) and (11) with SGD can be disrupted by a gradient explosion. Specifically, we limit the optimization in (10) and (11) subject to $\sigma_1(\mathbf{U}_h) < 2$, where $\sigma_i(.)$ represents the $i$-th largest singular value and $\mathbf{U}_h \in \mathbb{R}^{h\times h}$ denotes the weight matrix of the hidden state $\mathbf{h} \in \mathbb{R}^h$ with $h$ dimension in the GRU cell (see eq.(5) in [35] or (53) in appendix A.4.3).

**Theorem 3.** *When $\sigma_1(\mathbf{U}_h) < 2$, a GRU cell is locally stable at a fix point $\mathbf{h}^* = \mathbf{0}$.*

We use Lyapunov stability and two lemmas for the proof of **Theorem 3** in appendix A.4.3. Solving (10) and (11) with SGD gives an updated $\hat{\mathbf{U}}_h$ in each iteration that may not satisfy $\sigma_1(\hat{\mathbf{U}}_h) < 2$. We offer two solutions to satisfy the constraint outlined in theorem 3. The first solution relies on the spectral theorem, offering higher accuracy but with increased computational cost. Alternatively, the second solution employs the Gershgorin circle theorem, providing lower accuracy but at a reduced cost. Due to space limitations, we provide the details of the second solution in appendix A.4.5. For the first solution, we modify $\hat{\mathbf{U}}_h$ in three steps: (i) Decompose $\hat{\mathbf{U}}_h$ by singular value decomposition (SVD) such that $\hat{\mathbf{U}}_h = \mathbf{W}\mathbf{\Lambda}\mathbf{V}$. (ii) Replace the singular values of diagonal $\mathbf{\Lambda}$ that are greater than 2 with the threshold $2-\delta$ and obtain $\bar{\mathbf{\Lambda}} = \mathrm{Diag}(min(\sigma_1,2-\delta),...,min(\sigma_h,2-\delta))$. (iii) Reconstruct $\mathbf{U}_h \leftarrow \mathbf{W}\bar{\mathbf{\Lambda}}\mathbf{V}$. Employing SVD approach, the cost of this solution is proportional to $\mathcal{O}(h^3)$. In the next theorem we show that constructed $\mathbf{U}_h$ is the optimum solution. The proof is in appendix A.4.4.

**Theorem 4.** *The modified weight matrix $\mathbf{U}_h$ obtained from (iii) step above, is a solution of the following optimization problem:* $min_{\mathbf{U}_h}||\hat{\mathbf{U}}_h - \mathbf{U}_h||_F^2$, *s.t.* $\sigma_1(\mathbf{U}_h) < 2-\delta$.

## 7 Evaluation and Experiments

The first two experiments are single pendulum and double pendulum, where the dynamics of the latter one is more complicated. The third experiment is ball bouncing in irregular polygon to demonstrate the ability of the GIN when it faces irregular environments with multiple underlying dynamics. The fourth experiment covers visual odometry task for real world data. The last experiment shows the effectiveness of theorems 3 and 4 for gradient explosion handling. Intuitive python code and training algorithm are in appendix A.11. Detailed explanations regarding hyperparameter optimization, network structure, and empirical solutions aimed at avoiding poor local minima are in appendix A.7.

**Single Pendulum and Double Pendulum.**
In the pendulum experiment, observations are the image sequences comprising 100 time steps, each has the size of $24 \times 24$ distorted with time correlated noise. We perform the filtering-smoothing by the GIN. In state estimation task, the log-likelihood and

Table 1: Double (left) and single pendulum (middle) and polygon (left) state estimation results. $(s_1, s_2, s_3, s_4)$ are estimated position of the joints of double pendulum. For single pendulum and polygon, $(s_1, s_2)$ denotes the estimated single joint and ball position. $\mathbf{s}$ is sampled from eq. (10).

| Model | $SE^{Pos}_{s_1}$ | $SE^{Pos}_{s_3}$ | $SE^{Pos}_{s_2}$ | $SE^{Pos}_{s_4}$ | Log Likelihood | $SE^{Pos}_{s_1}$ | $SE^{Pos}_{s_2}$ | Log Likelihood | $SE^{Pos}_{s_1}$ | $SE^{Pos}_{s_2}$ | Log Likelihood |
|---|---|---|---|---|---|---|---|---|---|---|---|
| VAE | 0.212 | 0.275 | 0.192 | 0.285 | $1.953 \pm 0.306$ | 0.211 | 0.247 | $2.191 \pm 0.451$ | 0.769 | 0.782 | $2.094 \pm 0.392$ |
| IW-VAE | 0.203 | 0.251 | 0.201 | 0.266 | $2.113 \pm 0.391$ | 0.231 | 0.197 | $2.328 \pm 0.394$ | 0.777 | 0.764 | $2.117 \pm 0.475$ |
| VAE-RNN (LSTM) | 0.154 | 0.147 | 0.134 | 0.152 | $4.053 \pm 0.565$ | 0.111 | 0.099 | $5.691 \pm 0.151$ | 0.447 | 0.411 | $3.157 \pm 0.122$ |
| VAE-RNN (GRU) | 0.164 | 0.156 | 0.162 | 0.145 | $3.976 \pm 0.231$ | 0.109 | 0.101 | $5.749 \pm 0.168$ | 0.439 | 0.415 | $3.188 \pm 0.091$ |
| SVAE (LDS) | 0.190 | 0.211 | 0.179 | 0.159 | $3.495 \pm 0.425$ | 0.109 | 0.096 | $5.736 \pm 0.168$ | 0.424 | 0.438 | $3.229 \pm 0.112$ |
| KVAE | 0.193 | 0.188 | 0.178 | 0.149 | $3.679 \pm 0.101$ | 0.104 | 0.095 | $5.786 \pm 0.098$ | 0.434 | 0.439 | $3.194 \pm 0.073$ |
| EKVAE | 0.171 | 0.159 | 0.151 | 0.162 | $3.801 \pm 0.116$ | 0.088 | 0.093 | $5.858 \pm 0.113$ | 0.340 | 0.429 | $3.276 \pm 0.064$ |
| MVAE | 0.168 | 0.161 | 0.139 | 0.149 | $3.927 \pm 0.226$ | 0.087 | 0.087 | $5.938 \pm 0.137$ | 0.310 | 0.401 | $3.305 \pm 0.069$ |
| DeepAR | 0.175 | 0.189 | 0.157 | 0.147 | $3.646 \pm 0.294$ | 0.107 | 0.094 | $5.746 \pm 0.294$ | 0.447 | 0.434 | $3.071 \pm 0.106$ |
| RKN | 0.134 | 0.129 | 0.139 | 0.118 | $4.176 \pm 0.294$ | 0.078 | 0.075 | $6.161 \pm 0.23$ | 0.314 | 0.437 | $3.324 \pm 0.039$ |
| CRU | 0.117 | 0.127 | 0.116 | 0.104 | $4.269 \pm 0.237$ | 0.074 | 0.069 | $6.348 \pm 0.19$ | 0.294 | 0.378 | $3.366 \pm 0.039$ |
| Encode-Decoder | 0.291 | 0.284 | 0.342 | 0.317 | $1.877 \pm 0.427$ | 0.248 | 0.191 | $2.271 \pm 0.215$ | 0.969 | 0.914 | $1.984 \pm 0.384$ |
| AE-RNN (LSTM) | 0.163 | 0.171 | 0.148 | 0.167 | $3.901 \pm 0.706$ | 0.089 | 0.087 | $5.751 \pm 0.215$ | 0.349 | 0.319 | $3.331 \pm 0.124$ |
| AE-RNN (GRU) | 0.189 | 0.183 | 0.179 | 0.177 | $3.886 \pm 0.369$ | 0.091 | 0.085 | $5.798 \pm 0.205$ | 0.351 | 0.429 | $3.267 \pm 0.089$ |
| LGSSM$_{filter}$ | 0.125 | 0.119 | 0.121 | 0.107 | $4.192 \pm 0.127$ | 0.077 | 0.073 | $6.211 \pm 0.265$ | 0.448 | 0.312 | $3.171 \pm 0.365$ |
| LGSSM$_{smooth}$ | 0.109 | 0.111 | 0.104 | 0.101 | $4.231 \pm 0.154$ | 0.071 | 0.069 | $6.242 \pm 0.109$ | 0.327 | 0.317 | $3.544 \pm 0.029$ |
| GIN$_{filter}(n=2m)$ | 0.105 | 0.099 | 0.109 | 0.099 | $4.524 \pm 0.105$ | 0.063 | 0.06 | $7.192 \pm 0.239$ | 0.203 | 0.208 | $4.125 \pm 0.146$ |
| GIN$_{filter}(n=3m)$ | 0.083 | 0.081 | 0.088 | 0.079 | $4.829 \pm 0.151$ | 0.057 | 0.056 | $7.315 \pm 0.220$ | 0.198 | 0.199 | $4.254 \pm 0.187$ |
| GIN$_{smooth}(n=2m)$ | 0.081 | 0.094 | 0.081 | 0.082 | $4.708 \pm 0.123$ | 0.055 | 0.057 | $7.292 \pm 0.173$ | 0.186 | 0.198 | $4.227 \pm 0.048$ |
| GIN$_{smooth}(n=3m)$ | **0.069** | **0.073** | **0.075** | **0.067** | **$4.977 \pm 0.168$** | **0.049** | **0.047** | **$7.445 \pm 0.165$** | **0.154** | **0.147** | **$4.342 \pm 0.051$** |

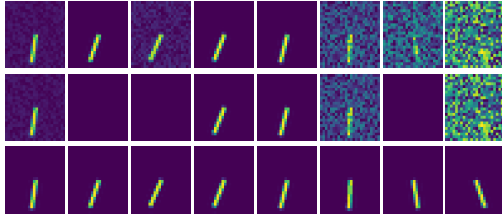

Figure 5: Pendulum imputation. Rows from up to down are the noise distorted ground truth, observation with uninformed missing frames and the imputation results of the GIN(smoothed).

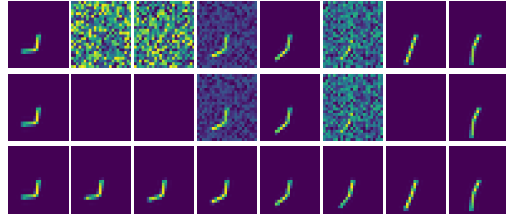

Figure 6: D-pendulum imputation. Rows from up to down are the noise distorted ground truth, observation with uninformed missing frames and the imputation results of the GIN(smoothed).

squared error (SE) of the estimated states are given in table 1. With $n = 3m$ ($n$ and $m$ representing state and transferred observation dimensions) as shown in Table 1, intuitively states contain information about position, velocity, and acceleration, increasing the likelihood compared to $n = 2m$, where only position and velocity are allocated to states. For the imputation task, we randomly remove half of the images from the generated sequences and perform image imputation by predicting the missing parts. The numerical results are in table 2 and imputed images are in figures 5 and 6. In table 2, the models with informed boolean masks are aware of available and missing images, while uninformed masks use a black image as input for missing images. This encourages the model to accurately infer the dynamics for image generation.

For ablation study, first we compare GIN with simple encoder-decoder without latent parameterization, and

Table 2: Image imputation task log likelihood for single pendulum (left), double pendulum (middle) and polygon (right). The approaches without good results are not included.

| Model | Single Pendulum | Double Pendulum | Irregular Polygon |
|---|---|---|---|
| VAE (informed) | $-36.751 \pm 7.227$ | $-44.264 \pm 4.232$ | |
| IW-VAE (informed) | $-34.241 \pm 5.421$ | $-43.728 \pm 5.421$ | |
| VAE-RNN (GRU) (informed) | $-14.243 \pm 0.328$ | $-15.983 \pm 0.425$ | $-16.483 \pm 0.425$ |
| SVAE (LDS) (informed smooth) | $-14.763 \pm 0.419$ | $-16.978 \pm 0.551$ | $-16.578 \pm 0.551$ |
| KVAE (informed smooth) | $-14.217 \pm 0.236$ | $-15.917 \pm 0.294$ | $-16.377 \pm 0.688$ |
| KVAE (unformed smooth) | $-39.260 \pm 5.399$ | $-38.544 \pm 6.419$ | |
| EKVAE (informed smooth) | $-12.897 \pm 0.524$ | $-13.917 \pm 0.414$ | $-15.916 \pm 0.292$ |
| EKVAE (unformed smooth) | $-29.246 \pm 3.328$ | $-33.548 \pm 4.516$ | |
| MKVAE (informed smooth) | $-12.723 \pm 0.428$ | $-13.889 \pm 0.298$ | $-15.698 \pm 0.344$ |
| MKVAE (unformed smooth) | $-21.524 \pm 1.003$ | $-26.773 \pm 1.537$ | $-31.634 \pm 2.816$ |
| DeepAR (informed) | $-14.019 \pm 0.486$ | $-15.881 \pm 0.511$ | $-19.322 \pm 0.418$ |
| RKN (informed) | $-12.782 \pm 0.016$ | $-13.832 \pm 0.023$ | $-16.267 \pm 0.313$ |
| RKN (unformed) | $-12.789 \pm 0.021$ | $-13.891 \pm 0.031$ | $-16.538 \pm 0.482$ |
| CRU (informed) | $-12.495 \pm 0.022$ | $-13.668 \pm 0.038$ | $-15.884 \pm 0.297$ |
| CRU (unformed) | $-12.554 \pm 0.029$ | $-13.739 \pm 0.061$ | $-15.972 \pm 0.394$ |
| Encoder-Decoder (informed) | $-64.569 \pm 9.146$ | $-75.179 \pm 21.527$ | |
| AE-RNN (GRU) (informed) | $-14.173 \pm 0.066$ | $-16.013 \pm 0.346$ | $-17.903 \pm 0.431$ |
| LGSSM (informed smooth) | $-12.395 \pm 0.048$ | $-13.775 \pm 0.013$ | $-15.831 \pm 0.215$ |
| GIN (informed smooth) | $-11.215 \pm 0.027$ | $-12.584 \pm 0.021$ | $-14.628 \pm 0.226$ |
| GIN (unformed smooth) | $-11.246 \pm 0.029$ | $-12.651 \pm 0.019$ | $-14.923 \pm 0.384$ |

then with LGSSM, where the GRU cells are omitted from the GIN structure and classic filtering-smoothing equations are used instead. We also check the usefulness of filtering-smoothing parameterization by replacing the GIN's transition block with GRU and LSTM cells and directly parameterize output distribution (like AE-RNN [36] and [37]). We compare various VI models, including VAE, IWVAE, VAE-RNN [38] which utilizes VAEs with recurrent cells, and EM-based VI approaches such as SVAE, KVAE, EKVAE and MVAE. Our comparison also encompasses DeepAR [39], an AR model. Our thorough comparison, covering the CRU, RKN, a group of VAEs, and AR models, enhances the depth of our analysis. The GIN consistently outperforms all other models. Additionally, we provide results using the MSE metric to highlight the competitive prediction accuracy of our approach (See appendix A.10).

**Bouncing Ball In Irregular Polygon.** To demonstrate the adaptability of the GIN, we subjected it to a more complex environment. We generated 1000 sequences, each comprising 70 time steps, depicting a ball moving within an irregular polygon featuring random shapes. The initial position and velocity of the ball were randomized. No external forces acted upon the ball, and collisions with the walls were elastic. In addition to the aforementioned models, we include latent SLDS structure from the SVAE approach for the comparison. SLDS is particularly relevant in scenarios such as this one, where separate dynamics for each state are considered, as seen when the ball bounces off the wall. We exclude simple baselines, such as encoder-decoder, VAE, and IW-VAE, from the comparison due to their lack of competitive performance. The numerical results are in tables 1 and 2. In Figure 7, we illustrate samples obtained from the trained smoothing distributions of GIN, LGSSM, and KVAE, showcasing their proximity to the ground truth. For this demonstration, we specifically select the 35th time step for sampling, denoted as $\mathcal{L}(\mathbf{s}_{35}|\mathbf{o}_{1:70})$. Here, $s1$ and $s2$ represent the estimated xy position of the ball within the polygon. Generating 20 frames with GIN in four different irregular environments is shown in figure

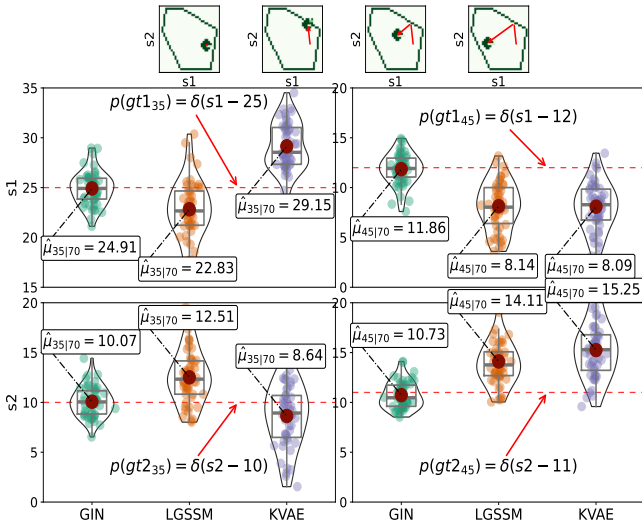

Figure 7: State (position) estimation in irregular polygon (6 edges) at 35-th and 45-th time steps. The upper row shows the ground truth ball position in 35, 39, 42 and 45-th time steps, respectively.

1. We have created animated files that demonstrate both sequence generation and imputation tasks within irregular polygons. Please visit: https://sites.google.com/view/ginneurips2024.

**Visual Odometry of KITTI Dataset.** We also evaluate the GIN with the higher dimensional observations for the visual odometry task on the KITTI dataset [40]. This dataset consists of 11 separated image sequences with their corresponding labels. We use a feature extractor network proposed by Zhou et al. in [41] to obtain the transferred observations of the GIN, i.e. $(\mathbf{w}, \mathbf{r})$. Additionally, we compare the results with AE-RNN(LSTM), AE-RNN(GRU), DeepVO [42] and KVAE. The numerical results are in table 3, where the common evaluation scheme for the KITTI dataset is exploited. The results of the KVAE degrades substantially as we have to reduce the size of the transferred observation to prevent the complexity of matrix inversion. Sampling visualization from smoothing state distribution are in appendix A.9.

Table 3: Comparison of model performance on KITTI dataset.

| Seq | AE-RNN(LSTM) | | AE-RNN(GRU) | | DeepVO | | KVAE | | LGSSM | | GIN | |
|------|---------------|----------------|---------------|----------------|----------------|----------------|----------------|----------------|----------------|----------------|----------------|----------------|
| | $t_{\mathrm{rel}}(\%)$ | $r_{\mathrm{rel}}(°)$ | $t_{\mathrm{rel}}(\%)$ | $r_{\mathrm{rel}}(°)$ | $t_{\mathrm{rel}}(\%)$ | $r_{\mathrm{rel}}(°)$ | $t_{\mathrm{rel}}(\%)$ | $r_{\mathrm{rel}}(°)$ | $t_{\mathrm{rel}}(\%)$ | $r_{\mathrm{rel}}(°)$ | $t_{\mathrm{rel}}(\%)$ | $r_{\mathrm{rel}}(°)$ |
| 03 | 8.99 | 4.55 | 9.34 | 3.81 | 8.49 | 6.89 | 12.14 | 4.38 | 7.51 | 3.98 | 6.98 | 3.27 |
| 04 | 11.88 | 3.44 | 12.36 | 2.89 | 7.19 | 6.97 | 13.17 | 4.73 | 9.12 | 2.64 | 9.14 | 2.28 |
| 05 | 8.96 | 3.43 | 10.02 | 3.43 | 2.62 | 3.61 | 11.47 | 5.14 | 6.11 | 3.21 | 4.38 | 2.51 |
| 06 | 9.66 | 2.8 | 10.99 | 3.22 | 5.42 | 5.82 | 10.93 | 3.98 | 6.70 | 3.51 | 6.14 | 2.90 |
| 07 | 9.83 | 5.48 | 13.70 | 6.52 | 3.91 | 4.60 | 12.73 | 4.68 | 6.59 | 3.49 | 7.21 | 2.98 |
| 10 | 13.58 | 3.49 | 13.37 | 3.25 | 8.11 | 8.83 | 14.79 | 10.91 | 9.32 | 2.90 | 8.37 | 2.59 |
| mean | 10.53 | 3.87 | 11.63 | 3.85 | 5.96 | 6.12 | 12.53 | 5.63 | 7.55 | 3.28 | 7.03 | 2.75 |

**Gradient Explosion Experiment.** Table 4 lists the log likelihood and its standard deviation for three experiments: single pendulum, double pendulum, and irregular polygon. These experiments were trained under different settings for gradient explosion handling: the conventional Gradient Clipping (GC), our first solution using Singular Value Decomposition (SVD), and our second solution using the Gershgorin Circle Theorem (GCT). The table empirically demonstrates that our method outperforms conventional gradient clipping. In this table, $\theta$ represents the threshold for gradient clipping as introduced in [10], and $\delta$ is the threshold in our method that keeps the spectral radius less than 2, i.e., $\sigma_1(\mathbf{U}_h) + \delta = 2$. As shown in table 4, gradient clipping failed to train for high $\theta$, while our approach achieves lower perplexity and higher log likelihood, ensuring stability compared to gradient clipping by constraining the GRU to be stable. We provide a comprehensive discussion on the $\delta$ variable, including how to tune it in different environments and its impact on model performance, in the appendix A.8.1.

Table 4: Gradient explosion handling: comparison between GC, SVD and GCT.

| | | Single Pendulum | | Double Pendulum | | Irregular Polygon | |
|---|---|---|---|---|---|---|---|
| | | Objective | Success | Objective | Success | Objective | Success |
| SVD (Ours) | $\delta = 0.1$ | $7.445 \pm 0.165$ | 100 % | $4.977 \pm 0.168$ | 100 % | $4.342 \pm 0.051$ | 100 % |
| | $\delta = 0.4$ | $7.281 \pm 0.149$ | 100 % | $4.847 \pm 0.241$ | 100 % | $4.191 \pm 0.094$ | 100 % |
| | $\delta = 0.7$ | $7.127 \pm 0.197$ | 100 % | $4.692 \pm 0.237$ | 100 % | $4.019 \pm 0.214$ | 100 % |
| | $\delta = 1$ | $6.935 \pm 0.271$ | 100 % | $4.471 \pm 0.341$ | 100 % | $3.839 \pm 0.207$ | 100 % |
| GCT (Ours) | $\delta = 0.1$ | $7.318 \pm 0.214$ | 100 % | $4.849 \pm 0.197$ | 100 % | $4.217 \pm 0.114$ | 100 % |
| | $\delta = 0.4$ | $7.184 \pm 0.170$ | 100 % | $4.711 \pm 0.179$ | 100 % | $4.079 \pm 0.124$ | 100 % |
| | $\delta = 0.7$ | $6.981 \pm 0.283$ | 100 % | $4.562 \pm 0.193$ | 100 % | $3.957 \pm 0.117$ | 100 % |
| | $\delta = 1$ | $6.787 \pm 0.321$ | 100 % | $4.291 \pm 0.281$ | 100 % | $3.672 \pm 0.176$ | 100 % |
| GC | $\theta = 5$ | $7.221 \pm 0.254$ | 100 % | $4.765 \pm 0.349$ | 100 % | $4.172 \pm 0.274$ | 100 % |
| | $\theta = 10$ | $7.301 \pm 0.495$ | 60 % | $4.834 \pm 0.592$ | 50 % | $4.222 \pm 0.570$ | 50 % |
| | $\theta = 15$ | $7.331 \pm 1.141$ | 20 % | N/A | 0 % | N/A | 0 % |
| | $\theta = 20$ | N/A | 0 % | N/A | 0 % | N/A | 0 % |

## 8   Conclusion

This paper introduces the GIN for representation learning in high-dimensional SSMs. The data flow is managed by Bayesian filtering-smoothing, and the use of GRU-based KG and SG networks addresses computational issues, resulting in an efficient and numerically stable model. The GIN learns system dynamics end-to-end, making it a high-performance model with strong system identification capabilities. It also provides insightful uncertainty representations for predictions and outperforms various counterparts, including generative models with variational inference, autoregressive models, and other baselines.

## 9   Acknowledgments

We thank Yasin Abbasi-Yadkori  for useful discussions and suggestions that contributed to this work.

## Bibliography

[1] C. M. Bishop and N. M. Nasrabadi, *Pattern recognition and machine learning*.   Springer, 2006, vol. 4, no. 4.

[2] R. M. Neal *et al.*, "Mcmc using hamiltonian dynamics," *Handbook of markov chain monte carlo*, vol. 2, no. 11, p. 2, 2011.

[3] D. P. Kingma and M. Welling, "Auto-encoding variational bayes," *arXiv preprint arXiv:1312.6114*, 2013.

[4] D. Koller and N. Friedman, *Probabilistic graphical models: principles and techniques*.   MIT press, 2009.

[5] R. E. Kalman, "A new approach to linear filtering and prediction problems," 1960.

[6] H. E. Rauch, F. Tung, and C. T. Striebel, "Maximum likelihood estimates of linear dynamic systems," *AIAA journal*, vol. 3, no. 8, pp. 1445–1450, 1965.

[7] E. A. Wan and R. Van Der Merwe, "The unscented kalman filter for nonlinear estimation," in *Proceedings of the IEEE 2000 Adaptive Systems for Signal Processing, Communications, and Control Symposium (Cat. No. 00EX373)*. Ieee, 2000, pp. 153–158.

[8] L. Ljung, "Asymptotic behavior of the extended kalman filter as a parameter estimator for linear systems," *IEEE Transactions on Automatic Control*, vol. 24, no. 1, pp. 36–50, 1979.

[9] S. Linderman, M. Johnson, A. Miller, R. Adams, D. Blei, and L. Paninski, "Bayesian learning and inference in recurrent switching linear dynamical systems," in *Artificial Intelligence and Statistics*. PMLR, 2017, pp. 914–922.

[10] R. Pascanu, T. Mikolov, and Y. Bengio, "On the difficulty of training recurrent neural networks," in *International conference on machine learning*. Pmlr, 2013, pp. 1310–1318.

[11] K. Doya *et al.*, "Bifurcations in the learning of recurrent neural networks 3," *learning (RTRL)*, vol. 3, p. 17, 1992.

[12] T. Haarnoja, A. Ajay, S. Levine, and P. Abbeel, "Backprop kf: Learning discriminative deterministic state estimators," in *Advances in neural information processing systems*, 2016, pp. 4376–4384.

[13] R. Krishnan, U. Shalit, and D. Sontag, "Structured inference networks for nonlinear state space models," in *Proceedings of the AAAI Conference on Artificial Intelligence*, vol. 31, no. 1, 2017.

[14] P. G. Chang, K. P. Murphy, and M. Jones, "On diagonal approximations to the extended kalman filter for online training of bayesian neural networks," in *Continual Lifelong Learning Workshop at ACML 2022*, 2022.

[15] M. Jones, T. R. Scott, M. Ren, G. F. Elsayed, K. Hermann, D. Mayo, and M. C. Mozer, "Learning in temporally structured environments," in *The Eleventh International Conference on Learning Representations*, 2022.

[16] M. Watter, J. T. Springenberg, J. Boedecker, and M. Riedmiller, "Embed to control: A locally linear latent dynamics model for control from raw images," *arXiv preprint arXiv:1506.07365*, 2015.

[17] Y. Burda, R. Grosse, and R. Salakhutdinov, "Importance weighted autoencoders," *arXiv preprint arXiv:1509.00519*, 2015.

[18] M. J. Johnson, D. K. Duvenaud, A. Wiltschko, R. P. Adams, and S. R. Datta, "Composing graphical models with neural networks for structured representations and fast inference," *Advances in neural information processing systems*, vol. 29, 2016.

[19] M. Fraccaro, S. Kamronn, U. Paquet, and O. Winther, "A disentangled recognition and nonlinear dynamics model for unsupervised learning," *arXiv preprint arXiv:1710.05741*, 2017.

[20] Y. Li and S. Mandt, "Disentangled sequential autoencoder," *arXiv preprint arXiv:1803.02991*, 2018.

[21] A. Klushyn, R. Kurle, M. Soelch, B. Cseke, and P. van der Smagt, "Latent matters: Learning deep state-space models," *Advances in Neural Information Processing Systems*, vol. 34, 2021.

[22] F. Tonolini, N. Aletras, Y. Jiao, and G. Kazai, "Robust weak supervision with variational auto-encoders," in *International Conference on Machine Learning*. PMLR, 2023, pp. 34 394–34 408.

[23] H. Zhu, C. Balsells-Rodas, and Y. Li, "Markovian gaussian process variational autoencoders," in *International Conference on Machine Learning*. PMLR, 2023, pp. 42 938–42 961.

[24] P. Becker, H. Pandya, G. Gebhardt, C. Zhao, C. J. Taylor, and G. Neumann, "Recurrent kalman networks: Factorized inference in high-dimensional deep feature spaces," in *International Conference on Machine Learning*. PMLR, 2019, pp. 544–552.

[25] M. Schirmer, M. Eltayeb, S. Lessmann, and M. Rudolph, "Modeling irregular time series with continuous recurrent units," in *International Conference on Machine Learning*. PMLR, 2022, pp. 19 388–19 405.

[26] S. Hochreiter and J. Schmidhuber, "Long short-term memory," *Neural computation*, vol. 9, no. 8, pp. 1735–1780, 1997.

[27] K. Cho, B. Van Merriënboer, D. Bahdanau, and Y. Bengio, "On the properties of neural machine translation: Encoder-decoder approaches," *arXiv preprint arXiv:1409.1259*, 2014.

[28] R. Wilson and L. Finkel, "A neural implementation of the kalman filter," *Advances in neural information processing systems*, vol. 22, pp. 2062–2070, 2009.

[29] I. Buchnik, G. Revach, D. Steger, R. J. Van Sloun, T. Routtenberg, and N. Shlezinger, "Latent-kalmannet: Learned kalman filtering for tracking from high-dimensional signals," *IEEE Transactions on Signal Processing*, 2023.

[30] G. Revach, N. Shlezinger, X. Ni, A. L. Escoriza, R. J. van Sloun, and Y. C. Eldar, "Kalman-net: Neural network aided kalman filtering for partially known dynamics," *arXiv preprint arXiv:2107.10043*, 2021.

[31] D. Ruhe and P. Forré, "Self-supervised inference in state-space models," *arXiv preprint arXiv:2107.13349*, 2021.

[32] T. Liu, K. A. Lee, Q. Wang, and H. Li, "Disentangling voice and content with self-supervision for speaker recognition," *Advances in Neural Information Processing Systems*, vol. 36, pp. 50 221–50 236, 2023.

[33] M. P. Deisenroth and H. Ohlsson, "A general perspective on gaussian filtering and smoothing: Explaining current and deriving new algorithms," in *Proceedings of the 2011 American Control Conference*. IEEE, 2011, pp. 1807–1812.

[34] P. Gilabert, G. Montoro, and E. Bertran, "On the wiener and hammerstein models for power amplifier predistortion," in *2005 Asia-Pacific Microwave Conference Proceedings*, vol. 2. IEEE, 2005, pp. 4–pp.

[35] J. Chung, C. Gulcehre, K. Cho, and Y. Bengio, "Empirical evaluation of gated recurrent neural networks on sequence modeling," *arXiv preprint arXiv:1412.3555*, 2014.

[36] N. Srivastava, E. Mansimov, and R. Salakhudinov, "Unsupervised learning of video representations using lstms," in *International conference on machine learning*. PMLR, 2015, pp. 843–852.

[37] H. Li, S. Yu, and J. Principe, "Causal recurrent variational autoencoder for medical time series generation," in *Proceedings of the AAAI Conference on Artificial Intelligence*, vol. 37, no. 7, 2023, pp. 8562–8570.

[38] J. Chung, K. Kastner, L. Dinh, K. Goel, A. C. Courville, and Y. Bengio, "A recurrent latent variable model for sequential data," *Advances in neural information processing systems*, vol. 28, 2015.

[39] D. Salinas, V. Flunkert, J. Gasthaus, and T. Januschowski, "Deepar: Probabilistic forecasting with autoregressive recurrent networks," *International Journal of Forecasting*, vol. 36, no. 3, pp. 1181–1191, 2020.

[40] A. Geiger, P. Lenz, and R. Urtasun, "Are we ready for autonomous driving? the kitti vision benchmark suite," in *2012 IEEE conference on computer vision and pattern recognition*. IEEE, 2012, pp. 3354–3361.

[41] T. Zhou, M. Brown, N. Snavely, and D. G. Lowe, "Unsupervised learning of depth and ego-motion from video," in *Proceedings of the IEEE conference on computer vision and pattern recognition*, 2017, pp. 1851–1858.

[42] S. Wang, R. Clark, H. Wen, and N. Trigoni, "Deepvo: Towards end-to-end visual odometry with deep recurrent convolutional neural networks," in *2017 IEEE international conference on robotics and automation (ICRA)*. IEEE, 2017, pp. 2043–2050.

[43] K. Sohn, H. Lee, and X. Yan, "Learning structured output representation using deep conditional generative models," *Advances in neural information processing systems*, vol. 28, 2015.

[44] D. Angeli, E. D. Sontag, and Y. Wang, "A characterization of integral input-to-state stability," *IEEE Transactions on Automatic Control*, vol. 45, no. 6, pp. 1082–1097, 2000.

[45] R. A. Horn and C. R. Johnson, *Matrix analysis*. Cambridge university press, 2012.

[46] D. P. Kingma and J. Ba, "Adam: A method for stochastic optimization," *arXiv preprint arXiv:1412.6980*, 2014.

[47] P. J. Werbos, "Backpropagation through time: what it does and how to do it," *Proceedings of the IEEE*, vol. 78, no. 10, pp. 1550–1560, 1990.

[48] J. L. Ba, J. R. Kiros, and G. E. Hinton, "Layer normalization," *arXiv preprint arXiv:1607.06450*, 2016.

[49] J. M. Wang, D. J. Fleet, and A. Hertzmann, "Gaussian process dynamical models for human motion," *IEEE transactions on pattern analysis and machine intelligence*, vol. 30, no. 2, pp. 283–298, 2007.

[50] J. Ko and D. Fox, "Learning gp-bayesfilters via gaussian process latent variable models," *Autonomous Robots*, vol. 30, no. 1, pp. 3–23, 2011.

[51] R. Frigola, F. Lindsten, T. B. Schön, and C. E. Rasmussen, "Bayesian inference and learning in gaussian process state-space models with particle mcmc," *Advances in neural information processing systems*, vol. 26, 2013.

[52] M. Schoukens and K. Tiels, "Identification of block-oriented nonlinear systems starting from linear approximations: A survey," *Automatica*, vol. 85, pp. 272–292, 2017.

[53] R. Li, S. John, and A. Solin, "Improving hyperparameter learning under approximate inference in gaussian process models," in *International Conference on Machine Learning*. PMLR, 2023, pp. 19 595–19 615.

[54] N. Wahlström, T. B. Schön, and M. P. Deisenroth, "From pixels to torques: Policy learning with deep dynamical models," *arXiv preprint arXiv:1502.02251*, 2015.

[55] E. Archer, I. M. Park, L. Buesing, J. Cunningham, and L. Paninski, "Black box variational inference for state space models," *arXiv preprint arXiv:1511.07367*, 2015.

[56] M. Karl, M. Soelch, J. Bayer, and P. Van der Smagt, "Deep variational bayes filters: Unsupervised learning of state space models from raw data," *arXiv preprint arXiv:1605.06432*, 2016.

[57] C. Naesseth, S. Linderman, R. Ranganath, and D. Blei, "Variational sequential monte carlo," in *International conference on artificial intelligence and statistics*. PMLR, 2018, pp. 968–977.

[58] S. S. Rangapuram, M. W. Seeger, J. Gasthaus, L. Stella, Y. Wang, and T. Januschowski, "Deep state space models for time series forecasting," *Advances in neural information processing systems*, vol. 31, 2018.

[59] V. Garcia Satorras, Z. Akata, and M. Welling, "Combining generative and discriminative models for hybrid inference," *Advances in Neural Information Processing Systems*, vol. 32, 2019.

# A Appendix

## A.1 Filtering and Smoothing Parameterization

The Kalman filter operates through two iterative steps: prediction and update. In the prediction step, the filter uses prior state information to make an estimate, while in the update step, it adjusts this estimate based on new observations. Assuming normally distributed additive process and observation noise, the filter can effectively perform these steps. During the prediction step, the filter employs the transition matrix $\mathbf{F}_{t+1}$ to estimate the next priors $p_{\gamma_{1:t+1}}(\mathbf{x}_{t+1}|\mathbf{w}_{1:t}) = (\mu_{t+1|t}, \boldsymbol{\Sigma}_{t+1|t})$, which represent the next state estimates without incorporating any new observations.

$$\mu_{t+1|t} = \mathbf{F}_{t+1}\mu_{t|t}, \text{and} \quad \boldsymbol{\Sigma}_{t+1|t} = \mathbf{F}_{t+1}\boldsymbol{\Sigma}_{t|t}\mathbf{F}_{t+1}^T + \mathbf{Q}_{t+1}, \text{and} \quad \mathbf{Q}_{t+1} = \sigma_{t+1}^2\mathbf{I} \tag{13}$$

When a new observation is available, the Kalman filter proceeds to the second step, where it updates the predicted prior using the new observation and the emission matrix $\mathbf{H}_{t+1}$. This results in the next posterior, $(\mu_{t+1|t+1}, \boldsymbol{\Sigma}_{t+1|t+1})$.

$$\mathbf{K}_{t+1} = \boldsymbol{\Sigma}_{t+1|t}\mathbf{H}_{t+1}^T\big(\mathbf{H}_{t+1}\boldsymbol{\Sigma}_{t+1|t}\mathbf{H}_{t+1}^T + \mathbf{R}_{t+1}\big)^{-1}, \tag{14}$$

$$\mu_{t+1|t+1} = \mu_{t+1|t} + \mathbf{K}_{t+1}(\mathbf{w}_t - \mathbf{H}_{t+1}\mu_{t+1|t}), \quad \boldsymbol{\Sigma}_{t+1|t+1} = \boldsymbol{\Sigma}_{t+1|t} - \mathbf{K}_{t+1}\big(\mathbf{H}_{t+1}\boldsymbol{\Sigma}_{t+1|t}\mathbf{H}_{t+1}^T + \mathbf{R}_{t+1}\big)\mathbf{K}_{t+1}. \tag{15}$$

The entire observation update procedure can be viewed as a weighted mean between the next prior, derived from the state update, and the new observation. This weighting is influenced by $\mathbf{Q}$ and $\mathbf{R}$, reflecting their inherent uncertainties. We derive the smoothing parameterization by leveraging the Markov property, which states that $\mathbf{x}_t$ is independent of future observations $\mathbf{w}_{t+1:T}$ given $\mathbf{x}_{t+1}$. Although $\mathbf{x}_{t+1}$ is unknown, there is a distribution over it. By conditioning on $\mathbf{x}_{t+1}$ and then marginalizing out, we can obtain $\mathbf{x}_t$ conditioned on $\mathbf{w}_{1:T}$.

$$
\begin{aligned}
p_{\gamma_{1:T}}(\mathbf{x}_t|\mathbf{w}_{1:T}) &= \int p_{\gamma_{1:T}}(\mathbf{x}_t|\mathbf{x}_{t+1}, \mathbf{w}_{1:T})p_{\gamma_{1:T}}(\mathbf{x}_{t+1}|\mathbf{w}_{1:T})d\mathbf{x}_{t+1} \\
&= \int p_{\gamma_{1:t}}(\mathbf{x}_t|\mathbf{x}_{t+1}, \mathbf{w}_{1:t}, \underline{\mathbf{w}_{t+1:T}})p_{\gamma_{1:T}}(\mathbf{x}_{t+1}|\mathbf{w}_{1:T})d\mathbf{x}_{t+1}
\end{aligned}
\tag{16}
$$

By using induction and and smoothed distribution for $t + 1$:

$$p_{\gamma_{1:T}}(\mathbf{x}_{t+1}|\mathbf{w}_{1:T}) = \mathcal{N}(\mu_{t+1|T}, \boldsymbol{\Sigma}_{t+1|T}) \tag{17}$$

we calculate the filtered two-slice distribution as follows:

$$\cdot p_{\gamma_{1:t}}(\mathbf{x}_t, \mathbf{x}_{t+1}|\mathbf{w}_{1:t}) = \mathcal{N}\left(\begin{pmatrix} \mu_{t|t} \\ \mu_{t+1|t} \end{pmatrix}, \begin{pmatrix} \boldsymbol{\Sigma}_{t|t} & \boldsymbol{\Sigma}_{t|t}\mathbf{F}_{t+1}^T \\ \mathbf{F}_{t+1}\boldsymbol{\Sigma}_{t|t} & \boldsymbol{\Sigma}_{t+1|t} \end{pmatrix}\right) \tag{18}$$

by using Gaussian conditioning we have:

$$p_{\gamma_{1:t}}(\mathbf{x}_t|\mathbf{x}_{t+1}, \mathbf{w}_{1:t}) = \mathcal{N}(\mu_{t|t} + \mathbf{J}_t(\mathbf{x}_{t+1} - \mathbf{F}_{t+1}\mu_{t|t}), \boldsymbol{\Sigma}_{t|t} - \mathbf{J}_t\boldsymbol{\Sigma}_{t+1|t}\mathbf{J}_t^T) \tag{19}$$

where $\mathbf{J}_t = \boldsymbol{\Sigma}_{t|t}\mathbf{F}_{t+1}[\boldsymbol{\Sigma}_{t+1|t}]^{-1}$. We calculate the smoothed distribution for $t$ using the rules of iterated expectation and covariance:

$$
\begin{aligned}
\mu_{t|T} &= \mathbb{E}\big[\mathbb{E}[\mathbf{x}_t|\mathbf{x}_{t+1}, \mathbf{w}_{1:T}] \,|\mathbf{w}_{1:T}\big] \\
&= \mathbb{E}\big[\mathbb{E}[\mathbf{x}_t|\mathbf{x}_{t+1}, \mathbf{w}_{1:t}] \,|\mathbf{w}_{1:T}\big] \\
&= \mathbb{E}\big[\mu_{t|t} + \mathbf{J}_t(\mathbf{x}_{t+1} - \mathbf{F}_{t+1}\mu_{t|t}) \,|\mathbf{w}_{1:T}\big] \\
&= \mu_{t|t} + \mathbf{J}_t(\mu_{t+1|T} - \mathbf{F}_{t+1}\mu_{t|t})
\end{aligned}
\tag{20}
$$

$$\Sigma_{t|T} = \text{cov}\big[\mathbb{E}[\mathbf{x}_t|\mathbf{x}_{t+1}, \mathbf{w}_{1:T}]\,|\mathbf{w}_{1:T}\big] + \mathbb{E}\big[\text{cov}[\mathbf{x}_t|\mathbf{x}_{t+1}, \mathbf{w}_{1:T}]\,|\mathbf{w}_{1:T}\big]$$

$$= \text{cov}\big[\mathbb{E}[\mathbf{x}_t|\mathbf{x}_{t+1}, \mathbf{w}_{1:t}]\,|\mathbf{w}_{1:T}\big] + \mathbb{E}\big[\text{cov}[\mathbf{x}_t|\mathbf{x}_{t+1}, \mathbf{w}_{1:t}]\,|\mathbf{w}_{1:T}\big]$$

$$= \text{cov}\big[\mu_{t|t} + \mathbf{J}_t(\mathbf{x}_{t+1} - \mathbf{F}_{t+1}\mu_{t|t})\,|\mathbf{w}_{1:T}\big] + \mathbb{E}\big[\Sigma_{t|t} - \mathbf{J}_t\Sigma_{t+1|t}\mathbf{J}_t^T\,|\mathbf{w}_{1:T}\big]$$

$$= \mathbf{J}_t\text{cov}\big[\mathbf{x}_{t+1} - \mathbf{F}_{t+1}\mu_{t|t}\,|\mathbf{w}_{1:T}\big]\mathbf{J}_t^T + \Sigma_{t|t} - \mathbf{J}_t\Sigma_{t+1|t}\mathbf{J}_t^T \tag{21}$$

$$= \mathbf{J}_t\Sigma_{t+1|T}\mathbf{J}_t^T + \Sigma_{t|t} - \mathbf{J}_t\Sigma_{t+1|t}\mathbf{J}_t^T$$

$$= \Sigma_{t|t} + \mathbf{J}_t\big(\Sigma_{t+1|T} - \Sigma_{t+1|t}\big)\mathbf{J}_t^T.$$

## A.2 Process Noise Matrix

As stated in (13), we can elaborate the process noise matrix at time $t$ in more details

$$\mathbf{Q}_t = \Sigma_{t|t-1} - \mathbf{F}_t\Sigma_{t-1|t-1}\mathbf{F}_t^T = \Sigma_{t|t-1} - \mathbf{F}_t\big[\Sigma_{t-1|t-2} - \mathbf{K}_{t-1}[\mathbf{H}_{t-1}\Sigma_{t-1|t-2}\mathbf{H}_{t-1}^T + \mathbf{R}_{t-1}]^{-1}\mathbf{K}_{t-1}^T\big]\mathbf{F}_t^T \tag{22}$$

combining (13) into (22) results in

$$\mathbf{Q}_t = \Sigma_{t|t-1} - \mathbf{F}_t\big[[\mathbf{F}_{t-1}\Sigma_{t-2|t-2}\mathbf{F}_{t-1}^T + \mathbf{Q}_{t-1}]$$

$$-\mathbf{K}_{t-1}[\mathbf{H}_{t-1}[\mathbf{F}_{t-1}\Sigma_{t-2|t-2}\mathbf{F}_{t-1}^T + \mathbf{Q}_{t-1}]\mathbf{H}_{t-1}^T + \mathbf{R}_{t-1}]^{-1}\mathbf{K}_{t-1}^T\big]\mathbf{F}_t^T \tag{23}$$

which is a function of $\mathbf{F}_t$, $\mathbf{Q}_{t-1}$, $\mathbf{F}_{t-1}$ and $\mathbf{H}_{t-1}$. In the GIN, $\hat{\mathbf{F}}_t$ and $\hat{\mathbf{H}}_t$ are learned by the *Dynamic Network* with the input of $\mathbf{x}_{t-1}$. From (15), $\mu_{t-1|t-1}$ is derived as a function of both $\mathbf{F}_{t-1}$ and $\mathbf{H}_{t-1}$, meaning the learned $\hat{\mathbf{F}}_t$ carries the information of both $\mathbf{H}_{t-1}$ and $\mathbf{F}_{t-1}$. Therefore, one can rewrite the equation (23) as

$$\mathbf{Q}_t = \mathbf{g}\bigg(\hat{\mathbf{F}}_t\big(\mu_{t-1|t-1}\big), \mathbf{Q}_{t-1}\bigg), \text{where} \quad \hat{\mathbf{F}}_t = \textit{Dynamic Net}\bigg(\mu_{t-1|t-1}\big(\mathbf{H}_{t-1}, \mathbf{F}_{t-1}\big)\bigg) \tag{24}$$

where $\mathbf{g}$ is a nonlinear function mapping $\mu_{t-1|t-1}$ and $\mathbf{Q}_{t-1}$ to $\mathbf{Q}_t$ and the graphical model for such choice of structure is in appendix figure 8b. It is possible to go one step further and simplify $\mu_{t-1|t-1}$ more, as it has $\Sigma_{\mathbf{t-1|t-1}}$ term in (15), combining it with (13) results in:

$$\mu_{t-1|t-1} = \mu_{t-1|t-2} + [\mathbf{F}_{t-1}\Sigma_{t-2|t-2}\mathbf{F}_{t-1}^T + \mathbf{Q}_{t-1}]$$

$$\mathbf{H}_{t-1}^T\big(\mathbf{H}_{t-1}[\mathbf{F}_{t-1}\Sigma_{t-2|t-2}\mathbf{F}_{t-1}^T + \mathbf{Q}_{t-1}]\mathbf{H}_{t-1}^T + \mathbf{R}_{t-1}\big)^{-1}(\mathbf{w}_t - \mathbf{H}_{t-1}\mu_{t-1|t-2}) \tag{25}$$

indicating that not only $\mathbf{F}_{t-1}$ and $\mathbf{H}_{t-1}$, but also $\mathbf{Q}_{t-1}$ is included in $\mu_{t-1|t-1}$, meaning that $\mathbf{Q}_t$ can be written solely as a function of $\mu_{t-1|t-1}$ and the graphical model for such choice is in figure 8a.

$$\mathbf{Q}_t = \mathbf{g}\bigg(\hat{\mathbf{F}}_t\big(\mu_{t-1|t-1}\big)\bigg), \text{where} \quad \hat{\mathbf{F}}_t = \textit{Dynamic Net}\bigg(\mu_{t-1|t-1}\big(\mathbf{H}_{t-1}, \mathbf{F}_{t-1}, \mathbf{Q}_{t-1}\big)\bigg). \tag{26}$$

We refer to $\mathbf{g}$ as the *Q Network*, which can be modeled either by an MLP, as shown in (26), or by a recurrent network, as detailed in (24).

## A.3 Output Distribution

In the case of grayscale images, each pixel $y$ is either one or zero with the probability $p$ or $1 - p$, respectively, meaning that $P(Y = y) = p^y(1 - p)^{1-y}$. This probability equation can be rewritten in the form of an exponential family as follows:

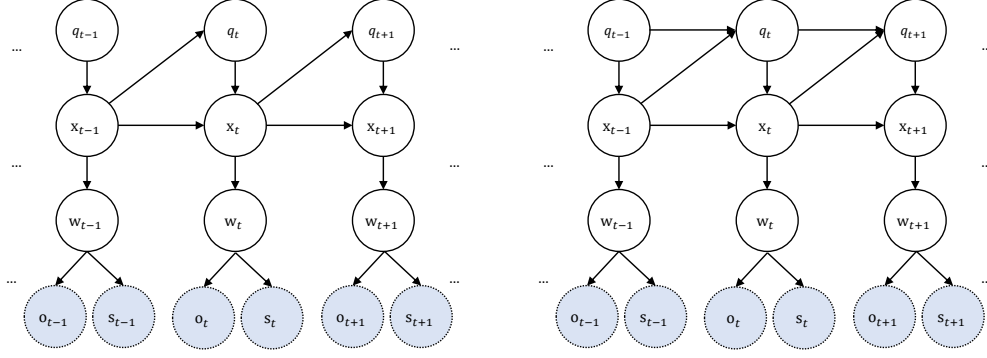

(a) Without recurrent dependency on $\mathbf{q}_t$ modeled with (26).

(b) With recurrent dependency on $\mathbf{q}_t$ modeled with (24).

Figure 8: Graphical models for different parameterizations of the process noise.

$$f_\theta(y) = h(y).exp\big(\theta.y - \psi(\theta)\big) \rightarrow e^{log(p^y(1-p)^{1-y})} = e^{y\,log(\frac{p}{1-p}) + log(1-p)}. \tag{27}$$

By choosing $\theta = log(\frac{p}{1-p})$ and $\psi(\theta) = log(1-p)$ and $h(y) = 1$, we can obtain $p = \frac{1}{1+e^{-\theta}}$. This means that by considering $\theta$ as the last layer of the decoder and applying a sigmoid layer, $p$ is obtained and one can optimize $p$ such that the likelihood is maximized as we did in in (11).

Similarly, consider $x$ as the ground truth state, $\hat{x}_\theta$ as the estimated state, and $\theta$ as the model variables. The residual follows a Gaussian distribution: $x = \hat{x}_\theta + \epsilon \sim \mathcal{N}(\hat{x}_\theta, \hat{\sigma}_\theta)$, where $\hat{\sigma}_\theta$ is the estimated variance. Then, the negative log likelihood is given by (28) as we obtained it in(10).

$$-log(\mathcal{L}) \propto \frac{1}{2}log(\hat{\sigma}_\theta) + \frac{(x - \hat{x}_\theta)^2}{2\hat{\sigma}_\theta} \tag{28}$$

## A.4 Proof of theorems

We restate the theorems for the ease of readability and then provide the proofs.

### A.4.1 Proof of theorem 1

*Theorem.* The log likelihood of the states given the original observations is determined as:

$$\mathcal{L}(\mathbf{s}_{1:T}|\mathbf{o}_{1:T}) = \frac{1}{N}\sum_{i=1}^{N}\log p(\mathbf{s}_{1:T}|\mathbf{x}_{1:T}^{(i)}, \mathbf{w}_{1:T}^{(i)}, \mathbf{o}_{1:T}) = \frac{1}{N}\sum_{i=1}^{N}\sum_{t=1}^{T}\log \mathcal{N}\left(\mathbf{s}_t \middle| d_\mathrm{m}(\hat{\mu}_{t|T}^{(i)}), d_\mathrm{c}(\hat{\mathbf{\Sigma}}_{t|T}^{(i)})\right)$$

where the $d_\mathrm{m}(.)$ and $d_\mathrm{c}(.)$ determines those parts of the decoder that obtain the state mean and variance, respectively. $N$ sequences of $(\mathbf{x}_{1:T}^{(i)}, \mathbf{w}_{1:T}^{(i)}) \sim q_{\gamma_{1:T}}(\mathbf{x}_{1:T}, \mathbf{w}_{1:T}|\mathbf{o}_{1:T})$ are sampled for Monte Carlo integration estimation.

*Proof.*

$$\log p_{\gamma_{1:T}}(\mathbf{s}_{1:T}|\mathbf{o}_{1:T})$$

$$= \int q_\phi(\mathbf{x}_{1:T}, \mathbf{w}_{1:T}|\mathbf{o}_{1:T}, \mathbf{s}_{1:T}) \log \frac{p_{\gamma_{1:T}}(\mathbf{s}_{1:T}, \mathbf{w}_{1:T}, \mathbf{x}_{1:T}|\mathbf{o}_{1:T})}{p_{\gamma_{1:T}}(\mathbf{x}_{1:T}, \mathbf{w}_{1:T}|\mathbf{s}_{1:T}, \mathbf{o}_{1:T})} \ d\mathbf{x}_{1:T} d\mathbf{w}_{1:T} \tag{29}$$

$$= \int q_\phi(\mathbf{x}_{1:T}, \mathbf{w}_{1:T}|\mathbf{o}_{1:T}, \mathbf{s}_{1:T}) \log q_\phi(\mathbf{x}_{1:T}, \mathbf{w}_{1:T}|\mathbf{o}_{1:T}, \mathbf{s}_{1:T})$$

$$- q_\phi(\mathbf{x}_{1:T}, \mathbf{w}_{1:T}|\mathbf{o}_{1:T}, \mathbf{s}_{1:T}) \log q_\phi(\mathbf{x}_{1:T}, \mathbf{w}_{1:T}|\mathbf{o}_{1:T}, \mathbf{s}_{1:T})$$

$$+ q_\phi(\mathbf{x}_{1:T}, \mathbf{w}_{1:T}|\mathbf{o}_{1:T}, \mathbf{s}_{1:T}) \log \frac{p_{\gamma_{1:T}}(\mathbf{s}_{1:T}, \mathbf{w}_{1:T}, \mathbf{x}_{1:T}|\mathbf{o}_{1:T})}{p_{\gamma_{1:T}}(\mathbf{x}_{1:T}, \mathbf{w}_{1:T}|\mathbf{s}_{1:T}, \mathbf{o}_{1:T})} \ d\mathbf{x}_{1:T} d\mathbf{w}_{1:T} \tag{30}$$

$$= \mathrm{KL}\big(q_\phi(\mathbf{x}_{1:T}, \mathbf{w}_{1:T}|\mathbf{o}_{1:T}, \mathbf{s}_{1:T})||p_{\gamma_{1:T}}(\mathbf{x}_{1:T}, \mathbf{w}_{1:T}|\mathbf{s}_{1:T}, \mathbf{o}_{1:T})\big)$$

$$+ \int q_\phi(\mathbf{x}_{1:T}, \mathbf{w}_{1:T}|\mathbf{o}_{1:T}, \mathbf{s}_{1:T}) \log \frac{p_{\gamma_{1:T}}(\mathbf{s}_{1:T}, \mathbf{w}_{1:T}, \mathbf{x}_{1:T}|\mathbf{o}_{1:T})}{q_\phi(\mathbf{x}_{1:T}, \mathbf{w}_{1:T}|\mathbf{o}_{1:T}, \mathbf{s}_{1:T})} \ d\mathbf{x}_{1:T} \mathbf{w}_{1:T} \tag{31}$$

$$\geq \mathbb{E}_{q_\phi(\mathbf{x}_{1:T}, \mathbf{w}_{1:T}|\mathbf{o}_{1:T}, \mathbf{s}_{1:T})}\Big[\log \frac{p_{\gamma_{1:T}}(\mathbf{s}_{1:T}, \mathbf{w}_{1:T}, \mathbf{x}_{1:T}|\mathbf{o}_{1:T})}{q_\phi(\mathbf{x}_{1:T}, \mathbf{w}_{1:T}|\mathbf{o}_{1:T}, \mathbf{s}_{1:T})}\Big] \tag{32}$$

$$= \mathbb{E}_{q_\phi(\mathbf{x}_{1:T}, \mathbf{w}_{1:T}|\mathbf{o}_{1:T}, \mathbf{s}_{1:T})}\Big[\log \frac{p_{\gamma_{1:T}}(\mathbf{s}_{1:T}|\mathbf{w}_{1:T}, \mathbf{x}_{1:T}, \mathbf{o}_{1:T}) . p_{\gamma_{1:T}}(\mathbf{w}_{1:T}, \mathbf{x}_{1:T}|\mathbf{o}_{1:T})}{q_\phi(\mathbf{x}_{1:T}, \mathbf{w}_{1:T}|\mathbf{o}_{1:T}, \mathbf{s}_{1:T})}\Big] \tag{33}$$

$$\approx \mathbb{E}_{q_\phi(\mathbf{x}_{1:T}, \mathbf{w}_{1:T}|\mathbf{o}_{1:T}, \mathbf{s}_{1:T})}\big[\log p_{\gamma_{1:T}}(\mathbf{s}_{1:T}|\mathbf{w}_{1:T}, \mathbf{x}_{1:T}, \mathbf{o}_{1:T})\big] \tag{34}$$

$$\approx \mathbb{E}_{q_{\gamma_{1:T}}(\mathbf{x}_{1:T}, \mathbf{w}_{1:T}|\mathbf{o}_{1:T})}\big[\log p_{\gamma_{1:T}}(\mathbf{s}_{1:T}|\mathbf{w}_{1:T}, \mathbf{x}_{1:T}, \mathbf{o}_{1:T})\big] \tag{35}$$

$$= \mathbb{E}_{q_{\gamma_{1:T}}(\mathbf{x}_{1:T}|\mathbf{w}_{1:T}) . p(\mathbf{w}_{1:T}|\mathbf{o}_{1:T})}\big[\log p_{\gamma_{1:T}}(\mathbf{s}_{1:T}|\mathbf{w}_{1:T}, \mathbf{x}_{1:T}, \mathbf{o}_{1:T})\big] \tag{36}$$

$$\approx \frac{1}{N} \sum_{i=1}^{N} \log p(\mathbf{s}_{1:T}|\mathbf{x}_{1:T}^{(i)}, \mathbf{w}_{1:T}^{(i)}, \mathbf{o}_{1:T}), \quad \mathbf{x}_{1:T}^{(i)} \sim q_{\gamma_{1:T}}(\mathbf{x}_{1:T}|\mathbf{w}_{1:T}^{(i)}), \quad \mathbf{w}_{1:T}^{(i)} \sim p(\mathbf{w}_{1:T}|\mathbf{o}_{1:T}) \tag{37}$$

$$= \frac{1}{N} \sum_{i=1}^{N} \sum_{t=1}^{T} \log \mathcal{N}\left(\mathbf{s}_t \Big| d_\mathrm{m}(\hat{\mu}_{t|T}^{(i)}), d_\mathrm{c}(\hat{\boldsymbol{\Sigma}}_{t|T}^{(i)})\right) \tag{38}$$

$$= \mathcal{L}(\mathbf{s}_{1:T}|\mathbf{o}_{1:T}). \tag{39}$$

The inequality in (32) arises from the positivity of the Kullback-Leibler (KL) term. The approximation in (34) results from assuming equality between the prior and the approximated posterior, i.e., $p_{\gamma_{1:T}}(\mathbf{w}_{1:T}, \mathbf{x}_{1:T}|\mathbf{o}_{1:T}) = q_\phi(\mathbf{x}_{1:T}, \mathbf{w}_{1:T}|\mathbf{o}_{1:T}, \mathbf{s}_{1:T})$. This assumption is not very restrictive as explained further in [43]. The approximation in (35) is coming from the equality assumption of $q_\phi(\mathbf{x}_{1:T}, \mathbf{w}_{1:T}|\mathbf{o}_{1:T}, \mathbf{s}_{1:T})$ and $q_{\gamma_{1:T}}(\mathbf{x}_{1:T}, \mathbf{w}_{1:T}|\mathbf{o}_{1:T})$. This assumption implies that given the original observation $\mathbf{o}_{1:T}$, our latent representation $(\mathbf{x}_{1:T}, \mathbf{w}_{1:T})$ is independent of the ground truth states $\mathbf{s}_{1:T}$. In simpler terms, $\mathbf{s}_{1:T}$ is concealed within $\mathbf{o}_{1:T}$, allowing us to disregard $\mathbf{s}_{1:T}$. The approximation in (37) is because of Monte Carlo integration estimation. $\qquad\square$

### A.4.2 Proof of theorem 2

*Theorem.* The (lower bound of) log likelihood of the original images is:

$$\mathcal{L}(\mathbf{o}_{1:T}) = \frac{1}{N} \sum_{i=1}^{N} \log p(\mathbf{o}_{1:T}|\mathbf{x}_{1:T}^{(i)}, \mathbf{w}_{1:T}^{(i)}) = \frac{1}{N} \sum_{i=1}^{N} \sum_{t=1}^{T} \sum_{k=1}^{D_o} \mathbf{o}_t^{(k)} \log\big(d_k(\hat{\mu}_{t|T}^{(i)})\big) + \big(1 - \mathbf{o}_t^{(k)}\big) \log(1 - d_k(\hat{\mu}_{t|T}^{(i)})) \tag{40}$$

with $N$ sequences of $(\mathbf{x}_{1:T}^{(i)}, \mathbf{w}_{1:T}^{(i)}) \sim q_{\gamma_{1:T}}(\mathbf{x}_{1:T}, \mathbf{w}_{1:T}|\mathbf{o}_{1:T})$ for Monte Carlo integration estimation. $d_k(.)$ defines the corresponding part of the decoder that maps the $k$-th pixel of $\mathbf{o}_t$ image.

*Proof.*

$$\log p_{\gamma_{1:T}}(\mathbf{o}_{1:T})$$

$$= \int q_{\gamma_{1:T}}(\mathbf{x}_{1:T}, \mathbf{w}_{1:T}|\mathbf{o}_{1:T}) \log \frac{p_{\gamma_{1:T}}(\mathbf{w}_{1:T}, \mathbf{x}_{1:T}, \mathbf{o}_{1:T})}{p_{\gamma_{1:T}}(\mathbf{x}_{1:T}, \mathbf{w}_{1:T}|\mathbf{o}_{1:T})} \, d\mathbf{x}_{1:T} d\mathbf{w}_{1:T} \tag{41}$$

$$= \int q_{\gamma_{1:T}}(\mathbf{x}_{1:T}, \mathbf{w}_{1:T}|\mathbf{o}_{1:T}) \log q_{\gamma_{1:T}}(\mathbf{x}_{1:T}, \mathbf{w}_{1:T}|\mathbf{o}_{1:T})$$
$$- q_{\gamma_{1:T}}(\mathbf{x}_{1:T}, \mathbf{w}_{1:T}|\mathbf{o}_{1:T}) \log q_{\gamma_{1:T}}(\mathbf{x}_{1:T}, \mathbf{w}_{1:T}|\mathbf{o}_{1:T})$$
$$+ q_{\gamma_{1:T}}(\mathbf{x}_{1:T}, \mathbf{w}_{1:T}|\mathbf{o}_{1:T}) \log \frac{p_{\gamma_{1:T}}(\mathbf{w}_{1:T}, \mathbf{x}_{1:T}, \mathbf{o}_{1:T})}{p_{\gamma_{1:T}}(\mathbf{x}_{1:T}, \mathbf{w}_{1:T}|\mathbf{o}_{1:T})} \, d\mathbf{x}_{1:T} d\mathbf{w}_{1:T} \tag{42}$$

$$= \mathrm{KL}\big(q_{\gamma_{1:T}}(\mathbf{x}_{1:T}, \mathbf{w}_{1:T}|\mathbf{o}_{1:T})||p_{\gamma_{1:T}}(\mathbf{x}_{1:T}, \mathbf{w}_{1:T}|\mathbf{o}_{1:T})\big)$$
$$+ \int q_{\gamma_{1:T}}(\mathbf{x}_{1:T}, \mathbf{w}_{1:T}|\mathbf{o}_{1:T}) \log \frac{p_{\gamma_{1:T}}(\mathbf{w}_{1:T}, \mathbf{x}_{1:T}, \mathbf{o}_{1:T})}{q_{\gamma_{1:T}}(\mathbf{x}_{1:T}, \mathbf{w}_{1:T}|\mathbf{o}_{1:T}} \, d\mathbf{x}_{1:T}\mathbf{w}_{1:T} \tag{43}$$

$$\geq \mathbb{E}_{q_{\gamma_{1:T}}(\mathbf{x}_{1:T}, \mathbf{w}_{1:T}|\mathbf{o}_{1:T})}\Big[\log \frac{p_{\gamma_{1:T}}(\mathbf{w}_{1:T}, \mathbf{x}_{1:T}, \mathbf{o}_{1:T})}{q_{\gamma_{1:T}}(\mathbf{x}_{1:T}, \mathbf{w}_{1:T}|\mathbf{o}_{1:T})}\Big] \tag{44}$$

$$= \mathbb{E}_{q_{\gamma_{1:T}}(\mathbf{x}_{1:T}, \mathbf{w}_{1:T}|\mathbf{o}_{1:T})}\Big[\log \frac{p_{\gamma_{1:T}}(\mathbf{o}_{1:T}|\mathbf{w}_{1:T}, \mathbf{x}_{1:T}).p_{\gamma_{1:T}}(\mathbf{w}_{1:T}, \mathbf{x}_{1:T})}{q_{\gamma_{1:T}}(\mathbf{x}_{1:T}, \mathbf{w}_{1:T}|\mathbf{o}_{1:T})}\Big] \tag{45}$$

$$\approx \mathbb{E}_{q_{\gamma_{1:T}}(\mathbf{x}_{1:T}, \mathbf{w}_{1:T}|\mathbf{o}_{1:T})}\Big[\log p_{\gamma_{1:T}}(\mathbf{o}_{1:T}|\mathbf{w}_{1:T}, \mathbf{x}_{1:T})\Big] \tag{46}$$

$$= \mathbb{E}_{q_{\gamma_{1:T}}(\mathbf{x}_{1:T}|\mathbf{w}_{1:T}).p(\mathbf{w}_{1:T}|\mathbf{o}_{1:T})}\Big[\log p_{\gamma_{1:T}}(\mathbf{o}_{1:T}|\mathbf{w}_{1:T}, \mathbf{x}_{1:T})\Big] \tag{47}$$

$$\approx \frac{1}{N}\sum_{i=1}^{N} \log p(\mathbf{o}_{1:T}|\mathbf{x}_{1:T}^{(i)}, \mathbf{w}_{1:T}^{(i)}), \quad \mathbf{x}_{1:T}^{(i)} \sim q_{\gamma_{1:T}}(\mathbf{x}_{1:T}|\mathbf{w}_{1:T}^{(i)}), \quad \mathbf{w}_{1:T}^{(i)} \sim p(\mathbf{w}_{1:T}|\mathbf{o}_{1:T}) \tag{48}$$

$$= \frac{1}{N}\sum_{i=1}^{N}\sum_{t=1}^{T}\sum_{k=1}^{D_o} \mathbf{o}_t^{(k)}\log\big(d_k(\hat{\mu}_{t|T}^{(i)})\big) + \big(1 - \mathbf{o}_t^{(k)}\big)\log(1 - d_k(\hat{\mu}_{t|T}^{(i)})) \tag{49}$$

$$= \mathcal{L}(\mathbf{o}_{1:T}). \tag{50}$$

The inequality in (44) arises from the positivity of the Kullback-Leibler (KL) term. The approximation in (46) results from assuming equality between the prior and the approximated posterior, i.e., $p_{\gamma_{1:T}}(\mathbf{w}_{1:T}, \mathbf{x}_{1:T}) = q_\phi(\mathbf{x}_{1:T}, \mathbf{w}_{1:T}|\mathbf{o}_{1:T})$. Although it is feasible to optimize (44) directly, we found out that by setting prior $p_{\gamma_{1:T}}(\mathbf{x}_{1:T}, \mathbf{w}_{1:T})$ equal to approximated posterior $q_{\gamma_{1:T}}(\mathbf{x}_{1:T}, \mathbf{w}_{1:T}|\mathbf{o}_{1:T})$, more stable results with concise improvement are achieved from the numerical results (similar to [19, 21, 43]). The approximation in (48) is because of Monte Carlo integration estimation.

$\square$

### A.4.3 Proof of theorem 3

First we provide a brief review on the GRU cell equations from [35]. Then we introduce two lemmas, by which we can proof theorem 2.

**GRU Cell Review.** The GRU cell holds the following equations:

$$\mathbf{z}_t = Sigmoid(\mathbf{U}_{xz}\mathbf{x}_t + \mathbf{U}_z\mathbf{h}_{t-1}), \tag{51}$$

$$\mathbf{r}_t = Sigmoid(\mathbf{U}_{xr}\mathbf{x}_t + \mathbf{U}_r\mathbf{h}_{t-1}), \tag{52}$$

$$\mathbf{h}_t = \mathbf{z}_t \odot \mathbf{h}_{t-1} + (1 - \mathbf{z}_t) \odot \hat{\mathbf{h}}_t, \tag{53}$$

$$\hat{\mathbf{h}}_t = tanh(\mathbf{U}_{xh}\mathbf{x}_t + \mathbf{U}_h(\mathbf{r}_t \odot \mathbf{h}_{t-1})) \tag{54}$$

where $\mathbf{x}_t$ and $\mathbf{h}_t$ are the input and the hidden state vectors, respectively. $\mathbf{U}_{xz}$, $\mathbf{U}_z$, $\mathbf{U}_{xr}$, $\mathbf{U}_r$, $\mathbf{U}_{xh}$ and $\mathbf{U}_h$ are all weight matrices.

**Lemma 1.** A GRU cell has a fixed point at $\mathbf{h}^* = \mathbf{0}$, if the input variable $\mathbf{x}_t = \mathbf{0}$.

*Proof.* After substituting $\mathbf{x}_t = \mathbf{0}$ and $\mathbf{h}_{t-1} = \mathbf{0}$ into (51) and (52), the update gate $\mathbf{z}_t$ and reset gate $\mathbf{r}_t$ are both $\frac{1}{2}$. Substituting these values along with $\mathbf{x} = \mathbf{0}$ and $\mathbf{h}_{t-1} = \mathbf{0}$ into (54) gives the candidate

state $\hat{\mathbf{h}}_t = \mathbf{0}$. Finally, substituting $\mathbf{h}_{t-1} = \mathbf{0}$, $\mathbf{z}_t = \frac{1}{2}$, and $\hat{\mathbf{h}}_t = \mathbf{0}$ into (53) results in the new state $\mathbf{h}_t = \mathbf{0}$. Thus, $\mathbf{h}_t = \mathbf{h}_{t-1} = \mathbf{0}$, indicating that GRU has a fixed point $\mathbf{h}^* = \mathbf{0}$. $\square$

**Lemma 2.** Let $\mathbf{I}$ be an $h \times h$ identity matrix, $\lambda_i(.)$ shows the $i$-th largest absolute eigenvalue, and $\mathbf{A} = \frac{1}{4}\mathbf{U}_h + \frac{1}{2}\mathbf{I}$. When the spectral radius $|\lambda_1(\mathbf{A})| < 1$, a GRU cell with input $\mathbf{x}_t = \mathbf{0}$. can be approximated by the following linearized GRU near $\mathbf{h}_t = \mathbf{0}$:

$$\mathbf{h}_t = \mathbf{A}\mathbf{h}_{t-1} \tag{55}$$

and the fixed point $\mathbf{h}^* = \mathbf{0}$ is locally stable.

*Proof.* By employing Taylor series expansion and linearizing $\mathbf{h}_t$ with respect to $\mathbf{h}_{t-1}$ at $\mathbf{h}_{t-1} = \mathbf{x}_t = \mathbf{0}$, $\mathbf{A}$ in (55) is obtain as $\frac{1}{4}\mathbf{U}_h + \frac{1}{2}\mathbf{I}$. Then, from (55) we get that $\mathbf{h}_t = \mathbf{A}^t\mathbf{h}_0$. Since $\mathbf{A}^t$ is dependent on the $t$-th powers of the eigenvalues of $\mathbf{A}$, the behavior of the linearized GRU is determined by the eigenvalues of $\mathbf{A}$. According to the Hartman-Grobman theorem, the behavior of a dynamical system near a hyperbolic fixed point is homeomorphic to the behavior of the linearized system. Therefore, when $|\lambda_1(\mathbf{A})| < 1$, the fixed point $\mathbf{h}^* = \mathbf{0}$ becomes a hyperbolic fixed point. Thus, a GRU cell can be approximated as $\mathbf{h}_t = \mathbf{A}^t\mathbf{h}_0$. Then, local stability is determined by the spectral radius $|\lambda_1(\mathbf{A})| < 1$ and then we have $|\mathbf{h}_t - \mathbf{h}_0|_{t\to\infty} = \mathbf{0}$ and the fixed point $\mathbf{h}^* = \mathbf{0}$ is locally stable. $\square$

**Theorem 3.** When $\sigma_1(\mathbf{U}_h) < 2$, a GRU cell is locally stable at a fix point $\mathbf{h}^* = \mathbf{0}$.

*Proof.* Relying on **Lemma1** and **Lemma2**, we need to satisfy $|\lambda_1(\frac{1}{4}\mathbf{U}_h + \frac{1}{2}\mathbf{I})| < 1$. $|\lambda_1(\frac{1}{4}\mathbf{U}_h + \frac{1}{2}\mathbf{I})| = |\frac{1}{4}\lambda_1(\mathbf{U}_h) + \frac{1}{2}|$ due to the properties of eigenvalues. From the triangle inequality we also have $|\frac{1}{4}\lambda_1(\mathbf{U}_h) + \frac{1}{2}| \leq \frac{1}{4}|\lambda_1(\mathbf{U}_h)| + \frac{1}{2}$. Using Weyl's inequality for its singular and eigenvalues, we have $|\lambda_1(\mathbf{U}_h)| \leq \sigma_1(\mathbf{U}_h)$ and therefore $\frac{1}{4}|\lambda_1(\mathbf{U}_h)| + \frac{1}{2} \leq \frac{1}{4}|\sigma_1(\mathbf{U}_h)| + \frac{1}{2}$. Thus, we need to satisfy $\frac{1}{4}|\sigma_1(\mathbf{U}_h)| + \frac{1}{2} < 1$ condition and conclude $\frac{1}{4}|\lambda_1(\mathbf{U}_h)| + \frac{1}{2} \leq \frac{1}{4}|\sigma_1(\mathbf{U}_h)| + \frac{1}{2} < 1$. This can be achieved by $\sigma_1(\mathbf{U}_h) < 2$.

The GRU cell is considered as a nonlinear dynamical system with $\dot{\mathbf{h}}_t = f(\mathbf{h}_t, \mathbf{x}_t)$ model, where one can drive $f$ from (51)-(54). We have shown the conditions for local stability of $\dot{\mathbf{h}} = f(\mathbf{h}, \mathbf{0})$ in the last paragraph. Now we show $\dot{\mathbf{h}} = f(\mathbf{h}, \mathbf{x})$ is locally stable if $\mathbf{x}_t \to \mathbf{0}$ as $t \to \infty$.

As we have the stability condition for $\dot{\mathbf{h}} = f(\mathbf{h}, \mathbf{0})$, there exists $\mathbf{V}(\mathbf{h})$ that is a Lyapunov function for this system such that:

$$\mathbf{V}(\mathbf{h}) \geq \mathbf{0}, \text{and } \dot{\mathbf{V}}(\mathbf{h}) = \frac{d\mathbf{V}}{d\mathbf{h}}.f(\mathbf{h}, \mathbf{0}) \leq -\mathbf{W}(\mathbf{h}) \tag{56}$$

for a positive definite $\mathbf{W}(\mathbf{h})$. When the input variable $\mathbf{x}$ is not zero, the system equation becomes $\dot{\mathbf{h}} = f(\mathbf{h}, \mathbf{x})$ and the derivative of $\mathbf{V}(\mathbf{h})$ becomes

$$\dot{\mathbf{V}}(\mathbf{h}) = \frac{d\mathbf{V}}{d\mathbf{h}}.f(\mathbf{h}, \mathbf{x}) \tag{57}$$

but for a small $\mathbf{x}_t$, we can write:

$$\dot{\mathbf{V}}(\mathbf{h}) \leq -\mathbf{W}(\mathbf{h}) + \epsilon, \tag{58}$$

$$\epsilon = |\frac{d\mathbf{V}}{d\mathbf{h}}.(f(\mathbf{h}, \mathbf{x}) - f(\mathbf{h}, \mathbf{0}))| \tag{59}$$

and $\epsilon \to 0$ as $\mathbf{x}_t \to \mathbf{0}$, utilizing Taylor series expansion of $f(\mathbf{h}, \mathbf{x})$ near $\mathbf{x} = \mathbf{0}$ [44]. Then, as $\mathbf{x}_t \to \mathbf{0}$, for any $\epsilon \geq 0$, there exists a $T$ such that for all $t \geq T$, $|\mathbf{x}_t| < \epsilon$ and therefore we have:

$$\dot{\mathbf{V}}(\mathbf{h}) \leq -\mathbf{W}(\mathbf{h}) + \epsilon, \tag{60}$$

for sufficiently small $\epsilon$, $\dot{\mathbf{V}}(\mathbf{h})$ remains negative.

Therefore, we need to demonstrate that the inputs of the GRU cells in (6) and (8) approach $\mathbf{0}$ as $t \to \infty$. As discussed in the main script, we apply a Wishart prior distribution on the covariance matrices, guiding them towards a scaled identity matrix. Over time, this scale diminishes towards zero, causing the covariance matrices—and consequently, the zero bias $\mathrm{Conv}(.)$ operators in (6) and

(8)—to converge to $\mathbf{0}$ as $t \to \infty$. This implies that the system tends to behave deterministically after prolonged observation, while the stability of the GRU cell is guaranteed. Informally, Theorem 3 requires us to asymptotically shrink the input covariance matrices of our GRU cells toward zero. While this shrinkage is reasonable as we expect the system to behave deterministically in the long run, it imposes a limitation on posterior inference. We acknowledge this as an open problem for future research. $\qquad\square$

### A.4.4 Proof of theorem 4

*Theorem.* The modified weight matrix $\mathbf{U}_h$ obtained from (iii) step above, is a solution of the following optimization problem: $\quad min_{\mathbf{U}_h}||\hat{\mathbf{U}}_h - \mathbf{U}_h||_F^2, \quad$ s.t. $\quad \sigma_1(\mathbf{U}_h) < 2 - \delta.$

*Proof.* Relying on [45], due to the matrices properties we have:

$$\sum_{i=1}^{h} \left[ \sigma_i(\hat{\mathbf{U}}_h) - \sigma_i(\mathbf{U}_h) \right]^2 \leq ||\hat{\mathbf{U}}_h - \mathbf{U}_h||_F^2. \tag{61}$$

Assuming that the first $s$ singular values of $\hat{\mathbf{U}}_h$ are greater than 2, we have $\sum_{i=1}^{h} \left[ \sigma_i(\hat{\mathbf{U}}_h) - \sigma_i(\mathbf{U}_h) \right] = \sum_{i=1}^{s} \left[ \sigma_i(\hat{\mathbf{U}}_h) - (2 - \delta) \right]$. According to (61), we need to show that $\sum_{i=1}^{s} \left[ \sigma_i(\hat{\mathbf{U}}_h) - (2 - \delta) \right]^2$ is equal to its upper bound $||\hat{\mathbf{U}}_h - \mathbf{U}_h||_F^2$, i.e. the optimum solution.

Using the singular value decomposition properties, we have $||\hat{\mathbf{U}}_h - \mathbf{U}_h||_F^2 = ||\mathbf{W}\mathbf{\Lambda}\mathbf{V} - \mathbf{W}\overline{\mathbf{\Lambda}}\mathbf{V}||_F^2 = ||\mathbf{W}(\mathbf{\Lambda} - \overline{\mathbf{\Lambda}})\mathbf{V}||_F^2 = tr(\mathbf{V}^*(\mathbf{\Lambda} - \overline{\mathbf{\Lambda}})\mathbf{W}^*\mathbf{W}(\mathbf{\Lambda} - \overline{\mathbf{\Lambda}})\mathbf{V}) = tr(\mathbf{V}^*(\mathbf{\Lambda} - \overline{\mathbf{\Lambda}})(\mathbf{\Lambda} - \overline{\mathbf{\Lambda}})\mathbf{V}) = tr(\mathbf{V}\mathbf{V}^*(\mathbf{\Lambda} - \overline{\mathbf{\Lambda}})(\mathbf{\Lambda} - \overline{\mathbf{\Lambda}})) = tr((\mathbf{\Lambda} - \overline{\mathbf{\Lambda}})(\mathbf{\Lambda} - \overline{\mathbf{\Lambda}})) = \sum_{i=1}^{s} \left[ \sigma_i(\hat{\mathbf{U}}_h) - (2 - \delta) \right]^2.$ $\qquad\square$

### A.4.5 Proof of theorem 5

*Theorem.* We can satisfy the constraint introduced in **Theorem 3** by applying the Gershgorin circle theorem as follows.

(i) In the $i$-th row of $\hat{\mathbf{U}}_h$, sum up the non diagonal elements as $i_d = \sum_{j=1, j \neq i}^{h} |\hat{\mathbf{U}}_h(i,j)|$ . Do (i) for the all rows.

(ii) For each $\hat{\mathbf{U}}_h(i,i)$ assign a circle cluster centered at $\hat{\mathbf{U}}_h(i,i)$ with $i_d$ radius in the complex plane.

(iii) Merge the clusters which have intersection and repeat step (iii) until no cluster can be merged any more. Finally the set of clusters $\mathcal{C} = \{\mathcal{C}_1, ..., \mathcal{C}_k\}$ with length of $k \leq h$ is obtained.

(iv) Define the radius of $\mathcal{C}_j$ cluster as $d_{\mathcal{C}_j} = max(\hat{\mathbf{U}}_h(i,i) + i_d | \forall \hat{\mathbf{U}}_h(i,i) \in \mathcal{C}_j)$. Calculate the radius of all $k$ clusters in step (iv).

(v) For each cluster $\mathcal{C}_j$, if $(\hat{\mathbf{U}}_h(i,i) + d_{\mathcal{C}_j} | \forall \hat{\mathbf{U}}_h(i,i) \in \mathcal{C}_j) > 2$, shift $\hat{\mathbf{U}}_h(i,i) \leftarrow (2 - d_{\mathcal{C}_j} - \delta)$ for a small $\delta$. Otherwise, keep $\hat{\mathbf{U}}_h(i,i)$ unchanged. Do step (v) for all clusters and their elements, i.e. $\hat{\mathbf{U}}_h(i,i)$.

These five steps, ensures the new $\hat{\mathbf{U}}_h$ follows the constraint introduced in **Theorem 3**.

*Proof.* The five steps elaborated above are direct results of the generalized Gershgorin circle theorem. This approach is faster than SVD method introduced in the paper, while it is not the best solution of the optimization problem introduced in the **Theorem 4**. $\qquad\square$

### A.5 Discussion About KL Term of Mode Collapse

The term introduced in the mode collapse handling section is:

$$-KL\big(p_i(\check{\mathbf{F}})||p_j(\check{\mathbf{F}}).pri_j(\check{\mathbf{F}})\big) + KL\big(p_i(\check{\mathbf{F}})||pri_i(\check{\mathbf{F}})\big). \tag{62}$$

Here $p_i(\check{\mathbf{F}}) \sim \mathcal{MN}_{n \times n}(\check{\mathbf{F}}^i, \mathbf{I}, \mathbf{I})$ and $p_j(\check{\mathbf{F}}) \sim \mathcal{MN}_{n \times n}(\check{\mathbf{F}}^j, \mathbf{I}, \mathbf{I})$, are matrix normal distributions with priors $pri_i(\check{\mathbf{F}}) \sim \mathcal{MN}_{n \times n}(\text{Diag}(\mathbf{m}_i), \mathbf{I}, \mathbf{I})$ and $pri_j(\check{\mathbf{F}}) \sim \mathcal{MN}_{n \times n}(\text{Diag}(\mathbf{m}_j), \mathbf{I}, \mathbf{I})$, respectively. We can expand (62) as:

$$= \int p_i(\check{\mathbf{F}}) \left( \log \frac{p_j(\check{\mathbf{F}}).pri_j(\check{\mathbf{F}})}{p_i(\check{\mathbf{F}})} + \log \frac{p_i(\check{\mathbf{F}})}{pri_i(\check{\mathbf{F}})} \right) \ d\check{\mathbf{F}} \tag{63}$$

$$= \int p_i(\check{\mathbf{F}}) \left( \log \frac{p_j(\check{\mathbf{F}}).pri_j(\check{\mathbf{F}})}{pri_i(\check{\mathbf{F}})} \right) \ d\check{\mathbf{F}} \tag{64}$$

$$= \int p_i(\check{\mathbf{F}}) \left( \log p_j(\check{\mathbf{F}}) \right) \ d\check{\mathbf{F}} + \int p_i(\check{\mathbf{F}}) \left( \log pri_j(\check{\mathbf{F}}) \right) \ d\check{\mathbf{F}} - \int p_i(\check{\mathbf{F}}) \left( \log pri_i(\check{\mathbf{F}}) \right) \ d\check{\mathbf{F}} \tag{65}$$

Relying on [45], we can simplify each term in (65) as:

$$\int p_i(\check{\mathbf{F}}) \left( \log p_j(\check{\mathbf{F}}) \right) \ d\check{\mathbf{F}} = -\frac{1}{2} \left[ \left( \text{vec}(\check{\mathbf{F}}^i) - \text{vec}(\check{\mathbf{F}}^j) \right)^T \left( \text{vec}(\check{\mathbf{F}}^i) - \text{vec}(\check{\mathbf{F}}^j) \right) + tr(\mathbf{I}) \right] + \text{const} \tag{66}$$

$$= -\frac{1}{2} ||\check{\mathbf{F}}^i - \check{\mathbf{F}}^j||_F^2 - \frac{1}{2} tr(\mathbf{I}) + \text{const} \tag{67}$$

$$\int p_i(\check{\mathbf{F}}) \left( \log pri_j(\check{\mathbf{F}}) \right) \ d\check{\mathbf{F}} = -\frac{1}{2} \left[ \left( \text{vec}(\check{\mathbf{F}}^i) - \text{vec}(\text{Diag}(\mathbf{m}_j)) \right)^T \left( \text{vec}(\check{\mathbf{F}}^i) - \text{vec}(\text{Diag}(\mathbf{m}_j)) \right) + tr(\mathbf{I}) \right] + \text{const} \tag{68}$$

$$= -\frac{1}{2} ||\check{\mathbf{F}}^i - \text{Diag}(\mathbf{m}_j)||_F^2 - \frac{1}{2} tr(\mathbf{I}) + \text{const} \tag{69}$$

$$\int p_i(\check{\mathbf{F}}) \left( \log pri_j(\check{\mathbf{F}}) \right) \ d\check{\mathbf{F}} = -\frac{1}{2} \left[ \left( \text{vec}(\check{\mathbf{F}}^i) - \text{vec}(\text{Diag}(\mathbf{m}_i)) \right)^T \left( \text{vec}(\check{\mathbf{F}}^i) - \text{vec}(\text{Diag}(\mathbf{m}_i)) \right) + tr(\mathbf{I}) \right] + \text{const} \tag{70}$$

$$= -\frac{1}{2} ||\check{\mathbf{F}}^i - \text{Diag}(\mathbf{m}_i)||_F^2 - \frac{1}{2} tr(\mathbf{I}). \tag{71}$$

Thus, we can rewrite (62) as:

$$\frac{1}{2} ||\check{\mathbf{F}}^i - \text{Diag}(\mathbf{m}_i)||_F^2 - \frac{1}{2} ||\check{\mathbf{F}}^i - \text{Diag}(\mathbf{m}_j)||_F^2 - \frac{1}{2} ||\check{\mathbf{F}}^i - \check{\mathbf{F}}^j||_F^2 - \frac{1}{2} tr(\mathbf{I}) + \text{const} \tag{72}$$

which is half of (12) plus a constant.

## A.6 Noise Generation Process

In our experiments, we use a time-correlated noise generation scheme to demonstrate the system's noise robustness. This method introduces a sequence of factors, $f_t$, of the same length as the data sequence, making the noise factors correlated over time. Let $f_0 \sim \mathcal{U}(0,1)$ and $f_{t+1} = \min(\max(0, f_t + r_t), 1)$ with $r_t \sim \mathcal{U}(-0.2, 0.2)$, where $f_0$ is the initialized factor and $\mathcal{U}$ is the uniform distribution. Two thresholds, $t_1 \sim \mathcal{U}(0, 0.25)$ and $t_2 \sim \mathcal{U}(0.75, 1)$, are defined. Values of $f_t$ less than $t_1$ are set to 0, values greater than $t_2$ are set to 1, and the remaining values are linearly scaled within the range $[0, 1]$. The original observation at time $t$, $\mathbf{o}_t$, is then given by $\mathbf{o}_t = f_t \mathbf{i}_t + (1 - f_t) \mathbf{i}_t^{pn}$, where $\mathbf{i}_t$ is the true image at time $t$ and $\mathbf{i}_t^{pn}$ is the pure noise generated for time $t$.

## A.7 Hyperparameters and training

In all experiments, we used the Adam optimizer [46] on an NVIDIA GeForce GTX 1050 Ti with 16GB RAM. To ensure optimal hyperparameters, we conducted a grid search. We searched for the initial learning rate in the range of $0.001$ to $0.2$ with increments of $0.005$, selecting the one that yielded the highest log-likelihood. We chose an initial learning rate of $0.006$ with an exponential decay rate of $0.9$ every 10 epochs. Backpropagation through time [47] was employed to compute gradients, given the use of GRU cells in our structure. The gradients are applied to GRU cells with the constraint explained in theorem 3, where we use the spectral theorem mentioned in the main script . We applied the layer normalization technique [48] to stabilize the dynamics within the recurrent structure and normalize the filter response. The Elu + 1 activation function was used to ensure the positivity of the diagonal elements of the process noise and covariance matrices.

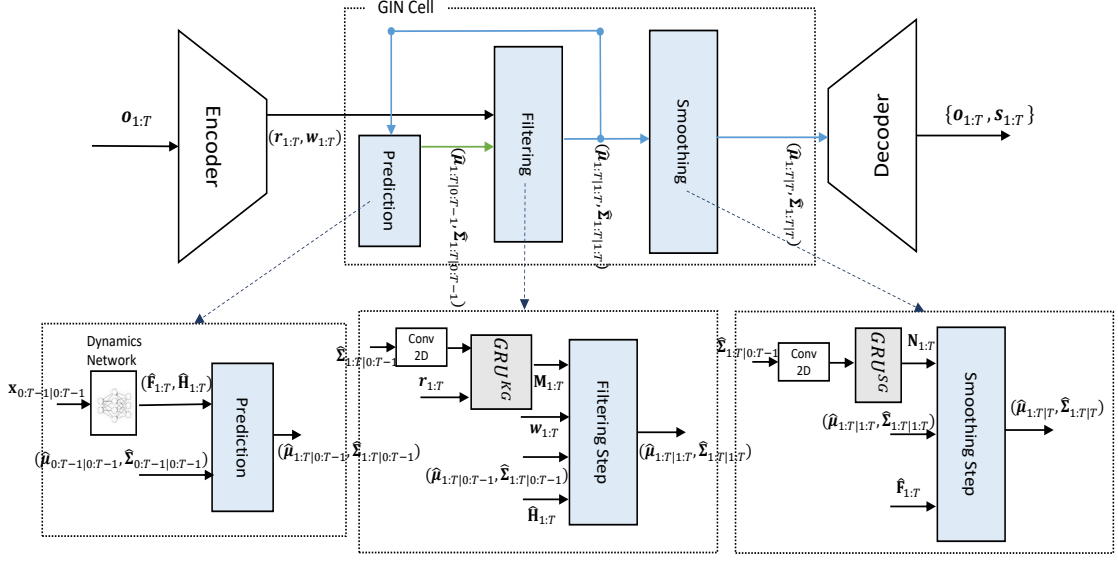

Figure 9: Proposed detailed architecture.

To prevent the model from getting stuck in poor local minima, such as focusing excessively on reconstruction rather than learning the dynamics obtained through filtering-smoothing, we find it beneficial to employ two training techniques for end-to-end learning:

1- Generating time correlated noisy sequences as consecutive observations, forces the model to learn the dynamics instead of focusing on reconstruction, e.g. figures 11 and 13 in appendix.

2- During the initial epochs, the system focuses on learning the auto-encoder and globally learned parameters, such as $\check{\mathbf{F}}^k$ and $\check{\mathbf{H}}^k$, while excluding the *Dynamic Net* parameters $\alpha_t(\mathbf{x}_{t-1})$. Afterwards, all parameters are jointly learned. This approach enables the system to initially acquire good embeddings and meaningful latent vectors before integrating the learning of $K$ different dynamics variables.

In our experiments, we set $K = 15$ to accommodate the complexity of the latent space, with each latent representing a unique dynamical scenario. Generally, parameter tuning is not challenging when the GIN is sufficiently flexible, as it can learn to prune unused elements through the *Dynamic Net*.

## A.8 Proposed architecture, qualitative comparison with the SOTA and empirical complexity analysis

### A.8.1 Architectures

The proposed detailed structure is shown in figure 9. To design the dynamic network, we use an MLP with 60 hidden units and a ReLU activation function, and a softmax activation in the last layer. The state mean, with a size of $n$, is the input to the dynamic network, which outputs $k$ coefficients. The structures of the encoder and decoder are shown in Table 5. In this table, $m$ represents the transferred observation dimension, with various values considered in the results. For state estimation tasks, the output size (*out*) is 4, 4, 8 and 12 for the single-pendulum, bouncing ball, double-pendulum and KITTI visual odometry experiments, respectively. For the imputation task, the number of hidden units in the KG and SG networks is set to 40 and 30, respectively. The convolutional layer applied over the covariance matrix has 8 filters with a kernel size of 5 and zero bias.

Table 5: The structure of the encoder and decoder.

| Encoder | Decoder |
|---|---|
| $6 \times 6$ Conv, 12, max pooling $2 \times 2$, stride $2 \times 2$<br>LayerNormalizer()<br>$4 \times 4$ Conv, 12, max pooling $2 \times 2$, stride $2 \times 2$<br>LayerNormalizer()<br>fully connected: 40<br>$\mathbf{w}$: fully connected: $m$, linear activation<br>$\mathbf{r}$: fully connected, Elu + 1 activation | if *state estimation task*:<br>  $\mathbf{s}_\mu$: fully connected: *out*, linear activation<br>  $\mathbf{s}_\Sigma$: fully connected, Elu + 1 zero bised<br>if *imputation task*:<br>  fully connected: 144<br>  $6 \times 6$ Trns Conv, 16, stride $4 \times 4$<br>  LayerNormalizer()<br>  $4 \times 4$ Trns Conv, 12, stride $2 \times 2$<br>  LayerNormalizer()<br>  $\mathbf{o}_i$: $1 \times 1$ Trns Conv, stride $1 \times 1$, softmax |

**Gradient Explosion Experiment.**

Referring to table 4, higher values of $\delta$ deteriorate performance because the convergence speed of the linearized GRU hidden state depends on $\sigma_1(\mathbf{U}_h)$ (see Eq. (55) in the appendix), which is projected to $2 - \delta$. Therefore, longer time dependencies in the data are eliminated for high $\delta$. On the other hand, smaller values of $\delta$ close to zero make the model more prone to noise [10]. Therefore, it is necessary to tune an appropriate $\delta$ within the range $0 < \delta < 2$, which is much easier than tuning the unbounded $\theta$.

### A.8.2 Empirical running times and parameters

We present the number of parameters for the utilized cell structures in our experiments and their corresponding empirical running times for 1 epoch in Table 6. In the first row of each model structure, we set the number of parameters approximately equal to our GIN to demonstrate the GIN's superior performance with the same parameter count. The extra running time of EM-variational approaches, like KVAE, is due to the use of classic Bayesian equations, which significantly increase running time for higher-dimensional observations. However, the GIN avoids this issue. The number of parameters in the GIN is noticeably lower than in other memory cells, such as LSTM and GRU, and EM-variational methods. This efficiency is achieved by converting high-dimensional sparse covariance matrices into lower-dimensional covariance matrices using a convolutional operator.

### A.8.3 Qualitative comparison of the GIN to recent related work.

Switching Linear Dynamical System models (SLDS) decompose complex, nonlinear time series data into sequences of simpler, reusable dynamical modes. Fitting an SLDS to data enables us to learn a flexible nonlinear generative model and parse data sequences into coherent discrete units. [9] is a SLDS model that incorporates additional latent variables to switch among different linear dynamics. However, this approach relies on Gibbs sampling for inferring the parameters, rendering it impractical for large datasets due to scalability issues.

Auto-regressive state space models (ARSSMs) are a class of dynamic models commonly used in time series analysis and forecasting. In these models, the evolution of a system over time is described by a set of states, by observing all data. Auto-Regressive (AR) Hidden Markov Models (AR-HMM) explain time series structures by defining a mapping from past observations to the current observation. [39] presents an ARHMM approach where target values are used directly as inputs. However, this reliance on target values makes the model more susceptible to noise.

Toward learning state space (system identification), a number of works including those by [49–53] have proposed algorithms for learning Gaussian Process SSMs (GPSSMs) through maximum likelihood estimation using the iterative EM algorithm. where filtering-smoothing with a set of fixed parameters $\gamma$ is conducted (E step), and then updating the set of parameters $\gamma$ is performed such that the obtained likelihood is maximized (M step). Frigola et al. [51] obtain sample trajectories from the smoothing distribution and then, conditioned on this trajectory, conduct the M step for the model's parameters. In the context of System Identification (SI), the GIN is akin to a Hammerstein-Wiener (HW) model [34], as it estimates system parameters directly from observations while transferring the observations through nonlinear functions.

Table 6: Empirical running times and parameters of experiments.

| Cell | Single Pend Param | Single Pend T/E | Double Pend Param | Double Pend T/E | Bouncing Ball Param | Bouncing Ball T/E | Visual Odometry Param | Visual Odometry T/E |
|---|---|---|---|---|---|---|---|---|
| LSTM ($n$=25) | ∼18k | ∼56s | ∼18k | ∼56s | ∼18k | ∼53s | ∼45k | ∼83s |
| LSTM ($n$=50) | ∼36k | ∼70s | ∼36k | ∼71s | ∼36k | ∼68s | ∼70k | ∼95s |
| LSTM ($n$=100) | ∼76k | ∼98s | ∼76k | ∼96s | ∼76k | ∼93s | ∼120k | ∼131s |
| GRU ($n$=30) | ∼18k | ∼61s | ∼18k | ∼62s | ∼18k | ∼59s | ∼42k | ∼79s |
| GRU ($n$=50) | ∼27k | ∼65s | ∼27k | ∼67s | ∼27k | ∼64s | ∼53k | ∼84s |
| GRU ($n$=100) | ∼57k | ∼86s | ∼57k | ∼85s | ∼57k | ∼84s | ∼90k | ∼111s |
| VAE ($n$=40) | ∼12k | ∼50s | ∼13k | ∼52s | ∼12k | ∼49s | ∼29k | ∼70s |
| IWVAE ($n$=40) | ∼13k | ∼50s | ∼13k | ∼54s | ∼11k | ∼47s | ∼30k | ∼75s |
| VAE-RNN ($n$=40) | ∼24k | ∼59s | ∼25k | ∼61s | ∼24k | ∼57s | ∼49k | ∼89s |
| SVAE ($n$=40) | ∼27k | ∼67s | ∼27k | ∼69s | ∼26k | ∼65s | ∼68k | ∼149s |
| KVAE ($n$=40) | ∼25k | ∼95s | ∼25k | ∼97s | ∼25k | ∼94s | ∼62k | ∼141s |
| KVAE ($n$=60) | ∼36k | ∼114s | ∼36k | ∼111s | ∼36k | ∼115s | ∼80k | ∼165s |
| EKVAE ($n$=40) | ∼26k | ∼98s | ∼26k | ∼99s | ∼26k | ∼94s | ∼65k | ∼145s |
| EKVAE ($n$=60) | ∼37k | ∼116s | ∼38k | ∼113s | ∼35k | ∼112s | ∼83k | ∼170s |
| MKVAE ($n$=40) | ∼34k | ∼112s | ∼34k | ∼110s | ∼33k | ∼105s | ∼77k | ∼153s |
| RKN ($n$=100) | ∼25k | ∼57s | ∼25k | ∼58s | ∼25k | ∼56s | ∼45k | ∼79s |
| CRU ($n$=100) | ∼24k | ∼55s | ∼24k | ∼55s | ∼23k | ∼54s | ∼46k | ∼78s |
| Encode-Decoder ($n$=30) | ∼10k | ∼45s | ∼10k | ∼44s | ∼9k | ∼44s | ∼21k | ∼63s |
| LGSSM ($n$=30) | ∼12k | ∼82s | ∼12k | ∼84s | ∼12k | ∼80s | ∼30k | ∼117s |
| LGSSM ($n$=45) | ∼15k | ∼98s | ∼15k | ∼97s | ∼15k | ∼97s | ∼36k | ∼136s |
| GIN ($n$=30) | ∼18k | ∼55s | ∼18k | ∼55s | ∼18k | ∼54s | ∼42k | ∼80s |
| GIN ($n$=45) | ∼25k | ∼59s | ∼25k | ∼58s | ∼25k | ∼57s | ∼48k | ∼83s |

In Table 7, we compare various algorithms to evaluate their ability to handle high-dimensional observations, learn dynamics, estimate state accurately, provide uncertainty estimates while handling noisy data, manage missing data, and perform direct optimization. Classic LGSSMs, such as the EKF and UKF, rely on the linearization of transition and emission equations and apply classic Bayesian updates over the linearized system with respect to the states. In other words, classic LGSSMs are model-based. In contrast, the GIN uses a data-driven network to learn parameters, offering a different approach from the traditional model-based methods.

Table 7: Learning the parameters in LGSSM is shown with ×/✓ because general LGSSMs, e.g. UKF and EKF, are not able to learn the parameters. However, in our setting and parameterization we use a data driven-based network for obtaining $(\mathbf{F}, \mathbf{H})$ to make LGSSMs comparable with the GIN for high dimensional observation experiments.

| Model | high-d | learn dynamics | missing-imputation | state est | uncertainty | noise handling | dir opt |
|---|---|---|---|---|---|---|---|
| LSTM [26] | ✓ | ✓ | ✓ | ✓ | × | ✓ | ✓ |
| GRU [27] | ✓ | ✓ | ✓ | ✓ | × | ✓ | ✓ |
| P2T [54] | ✓ | ✓ | × | ✓ | × | × | ✓ |
| E2C [16] | ✓ | ✓ | × | × | ✓ | × | × |
| BB-VI [55] | × | ✓ | ✓ | × | ✓ | ✓ | × |
| SIN [13] | ✓ | ✓ | ✓ | × | ✓ | ✓ | × |
| DVBF [56] | ✓ | ✓ | ✓ | × | ✓ | ✓ | × |
| VSMC [57] | ✓ | ✓ | ✓ | × | ✓ | ✓ | × |
| DSA [20] | ✓ | ✓ | × | × | ✓ | × | × |
| KVAE [19] | × | ✓ | ✓ | ✓ | ✓ | ✓ | × |
| EKVAE [21] | × | ✓ | ✓ | ✓ | ✓ | ✓ | × |
| MVAE [23] | × | ✓ | ✓ | ✓ | ✓ | ✓ | × |
| rSLSD [9] | × | × | × | ✓ | ✓ | × | × |
| DeepAR [39] | × | ✓ | × | ✓ | ✓ | × | ✓ |
| DSSM [58] | × | ✓ | × | ✓ | ✓ | × | ✓ |
| HybridGNN [59] | × | × | ✓ | ✓ | × | ✓ | ✓ |
| KalmanNet [30] | × | × | ✓ | ✓ | × | ✓ | ✓ |
| SSI [31] | × | × | ✓ | ✓ | ✓ | ✓ | ✓ |
| LGSSM | × | ×/✓ | ✓ | ✓ | ✓ | ✓ | ✓ |
| GIN | ✓ | ✓ | ✓ | ✓ | ✓ | ✓ | ✓ |

## A.9 Visualization and The Imputation

Graphical results of informed, uninformed and noisy observations for image imputation task for both single and double pendulum experiments can be found in 10, 11, 12 and 13 figures. Inference for the trained smoothing distribution of all experiments are in 14, 15, 16, , 17, 18, 19, 20, 21, 22, 23 and 24 figures, where we generated samples from the smoothing estimated states distribution, $(\mathbf{s}_t|\mathbf{o}_{1:T})$. Then we fit density on the generated samples. This visualization shows the effectiveness of the GIN in reducing the uncertainty of the estimates compare to LGSSM and KVAE.

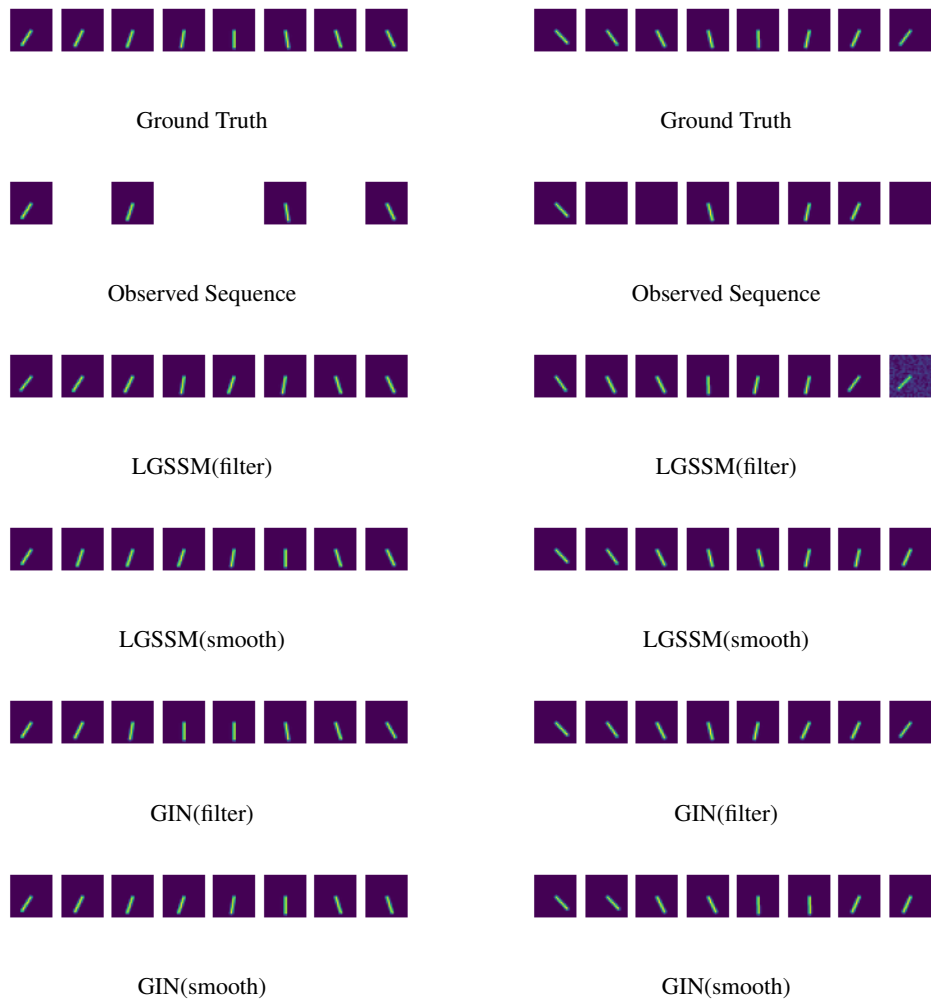

Figure 10: Informed(left column) and uninformed(right column) image imputation task for the single pendulum experiments.

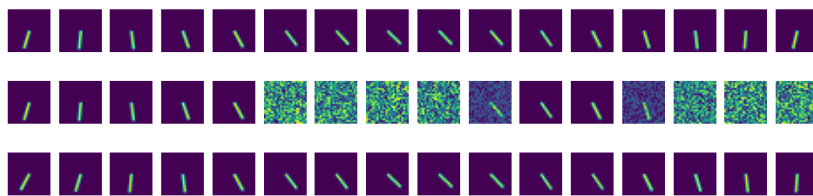

Figure 11: Image imputation task for the single pendulum experiment exposed to the noisy observations, where the generated noise has correlation with the time. Each figure, beginning from top to bottom, indicates the ground truth, noisy observation and the imputation results of the GIN.

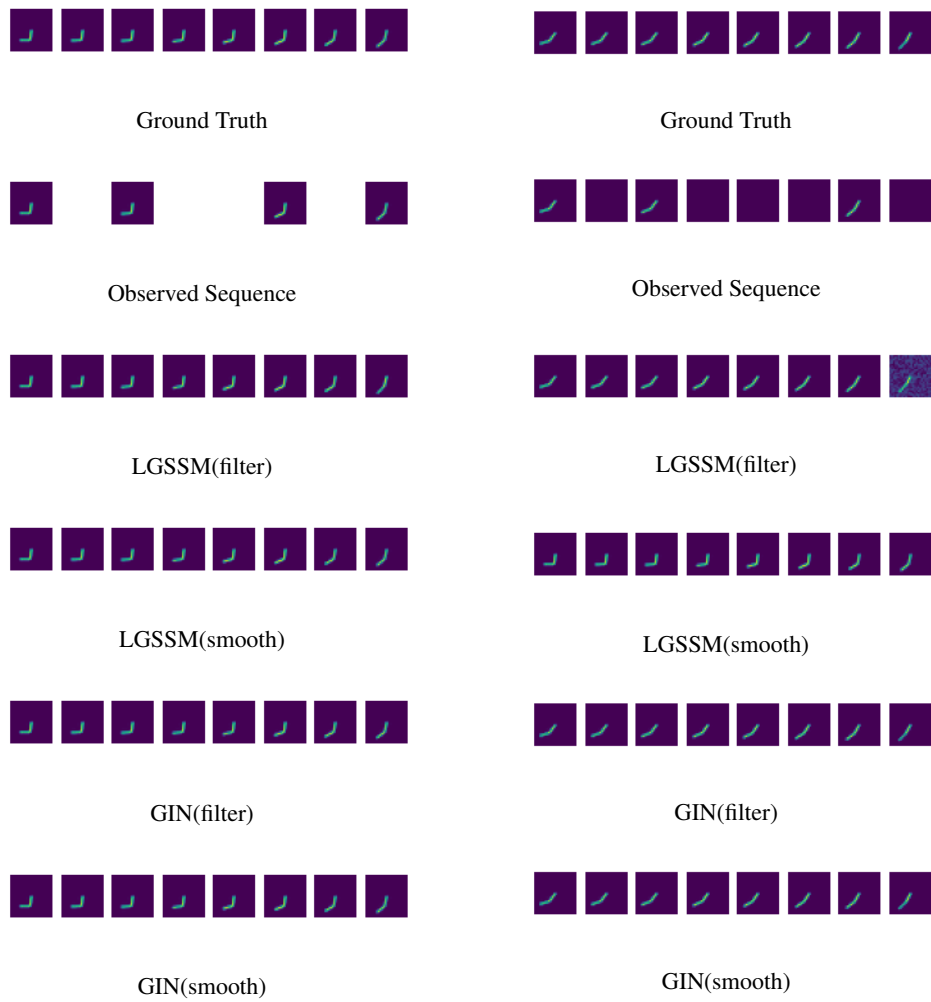

Figure 12: Informed(left column) and uninformed(right column) image imputation task for the double pendulum experiments.

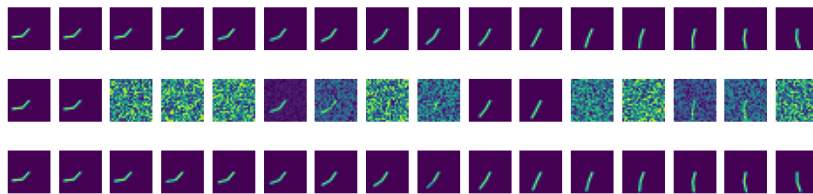

Figure 13: Image imputation task for the double pendulum experiment exposed to the noisy observations, where the generated noise has correlation with the time. Each figure, beginning from top to bottom, indicates the ground truth, noisy observation and the imputation results of the GIN.

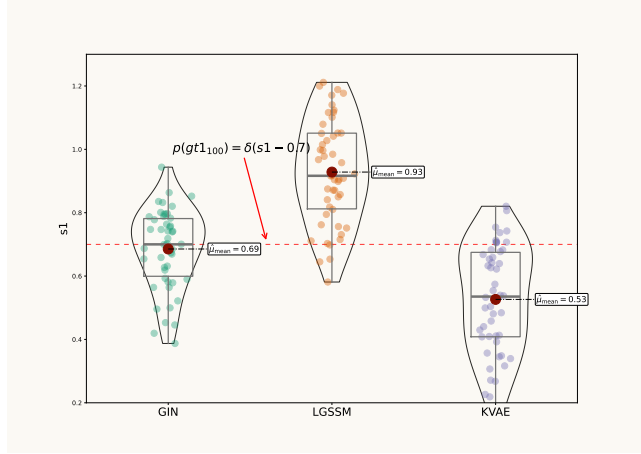

Figure 14: Inference for the single pendulum $s1$ position at 100-th time step. Generated samples from smoothened distribution, $(s1_{100}|\mathbf{o}_{1:150})$, trained by the GIN, LGSSM and KVAE, respectively. The dashed red line $p(gt1_{100})$ is the ground truth state with distribution of $\delta(s1_{100}-0.7)$. We calculate the sample mean and fit a distribution on the samples for further visualization and comparison purpose.

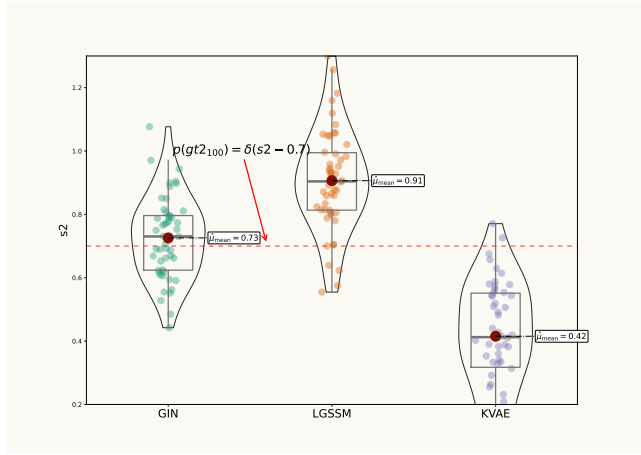

Figure 15: Inference for the single pendulum $s2$ position at 100-th time step. Generated samples from smoothened distribution, $(s2_{100}|\mathbf{o}_{1:150})$, trained by the GIN, LGSSM and KVAE, respectively.

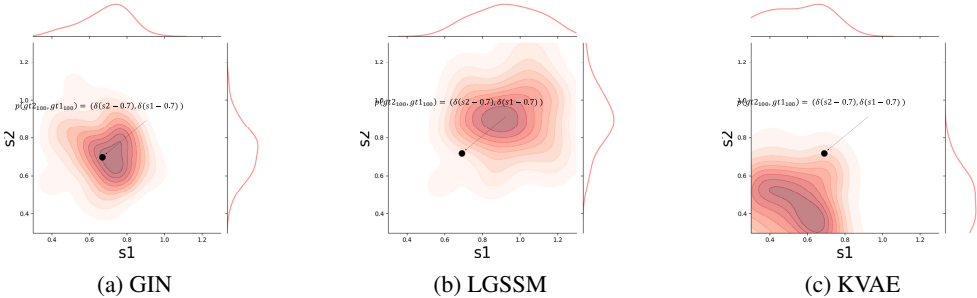

(a) GIN        (b) LGSSM        (c) KVAE

Figure 16: Generated samples from the trained smoothened joint distribution of the single pendulum position, $(s1, s2)$, at 100-th time step for the GIN, LGSSM and KVAE, respectively. The ground truth is shown with a black point.

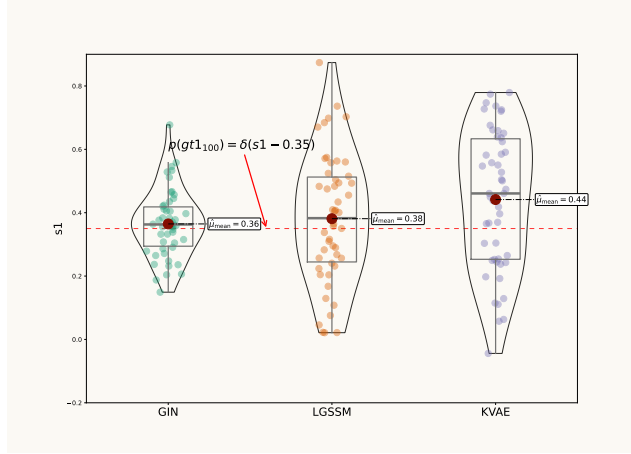

Figure 17: Inference for the double pendulum $s1$ position at 100-th time step. Generated samples from smoothened distribution, $(s1_{100}|\mathbf{o}_{1:150})$, trained by the GIN, LGSSM and KVAE, respectively. The dashed red line $p(gt1_{100})$ is the ground truth state with distribution of $\delta(s1_{100} - 0.35)$.

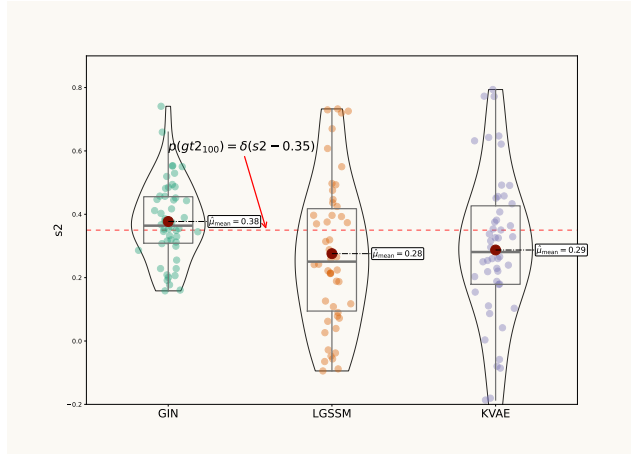

Figure 18: Inference for the double pendulum $s2$ position at 100-th time step. Generated samples from smoothened distribution, $(s2_{100}|\mathbf{o}_{1:150})$, trained by the GIN, LGSSM and KVAE, respectively. The dashed red line $p(gt2_{100})$ is the ground truth state with distribution of $\delta(s2_{100} - 0.35)$.

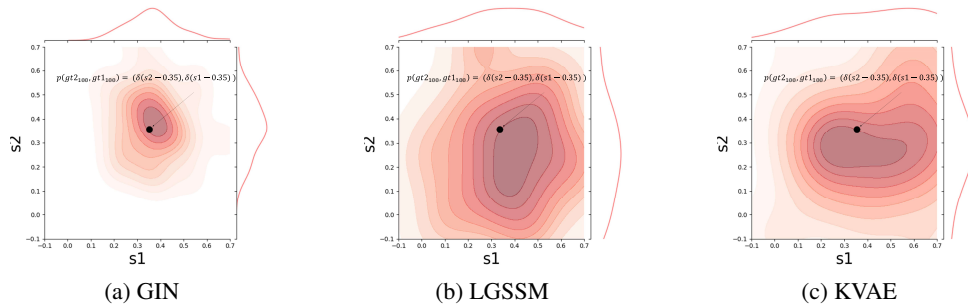

(a) GIN        (b) LGSSM        (c) KVAE

Figure 19: Generated samples from the trained smoothened joint distribution of the double pendulum first joint position, $(s1, s2)$, at 100-th time step for the GIN, LGSSM and KVAE, respectively. The ground truth is shown with a black point.

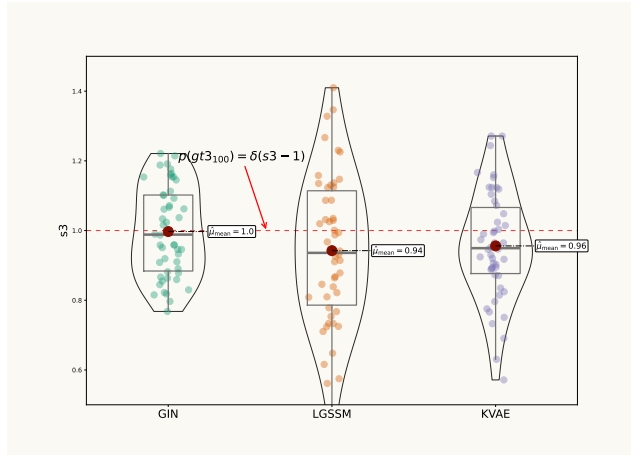

Figure 20: Inference for the double pendulum $s3$ position at 100-th time step. Generated samples from smoothened distribution, $(s3_{100}|\mathbf{o}_{1:150})$, trained by the GIN, LGSSM and KVAE, respectively. The dashed red line $p(gt3_{100})$ is the ground truth state with distribution of $\delta(s3_{100} - 1)$.

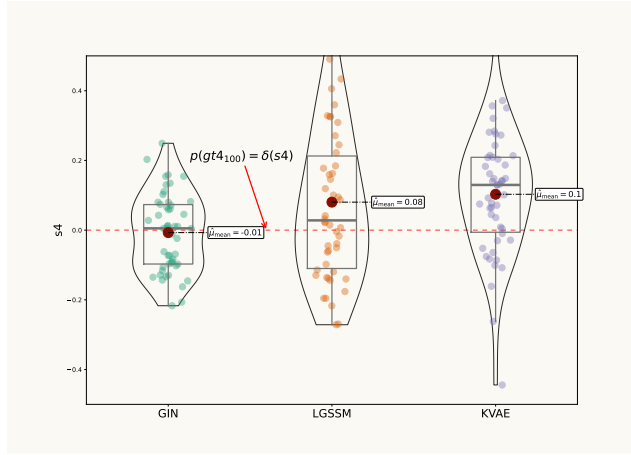

Figure 21: Inference for the double pendulum $s4$ position at 100-th time step. Generated samples from smoothened distribution, $(s4_{100}|\mathbf{o}_{1:150})$, trained by the GIN, LGSSM and KVAE, respectively.

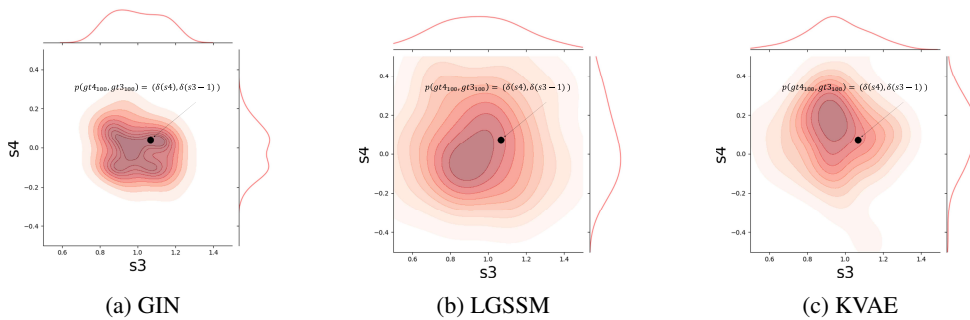

(a) GIN $\qquad$ (b) LGSSM $\qquad$ (c) KVAE

Figure 22: Generated samples from the trained smoothened joint distribution of the double pendulum second joint position, $(s3, s4)$, at 100-th time step for the GIN, LGSSM and KVAE, respectively. The ground truth is shown with a black point.

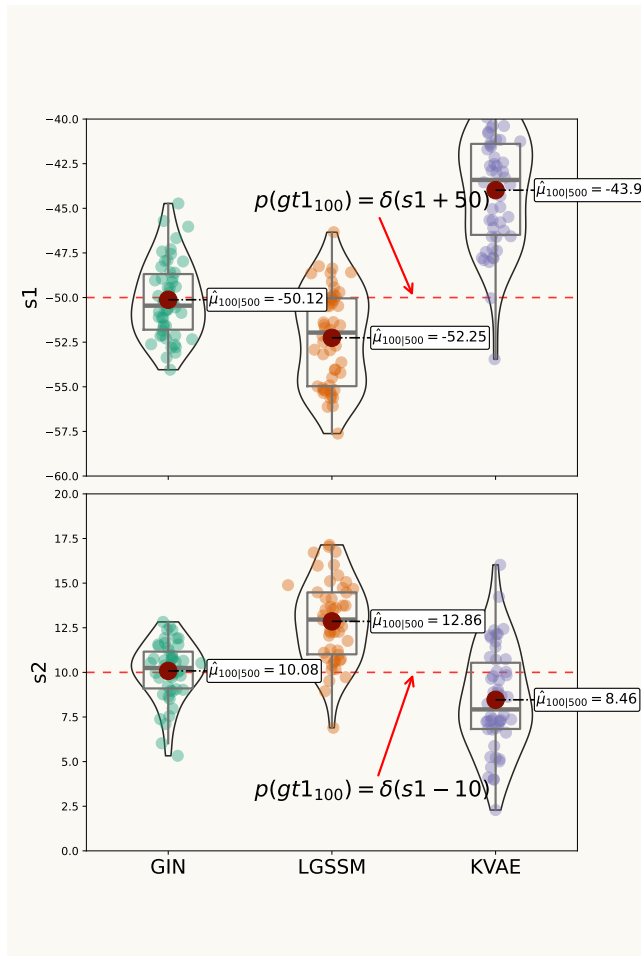

Figure 23: Inference for the visual odometry $s1$ and $s2$ positions at 100-th time step. Generated samples from smoothened distribution, $(s1_{100}, s2_{100}|\mathbf{o}_{1:500})$, trained by the GIN, LGSSM and KVAE, respectively. The dashed red line $p(gt1_{100}, gt2_{100})$ is the ground truth state with distribution of $(\delta(s1_{100} + 50), \delta(s2_{100} - 10))$.

## A.10  MSE Results for The State Estimation and Estimated KG-SG

The MSE results for the single pendulum, double pendulum, and bouncing ball experiments are shown in Tables 8 and 9 and 10. In addition to the dynamics equation (5), where the $\hat{\mathbf{F}}$ matrix includes the effects of the process noise, two other solutions introduced in Section 5 are included in the MSE results. Parameterizing the process noise as in (24) with a GRU cell is indicated by GRU($\mathbf{Q}$), while another parameterization, shown as MLP($\mathbf{Q}$), is done using (26).

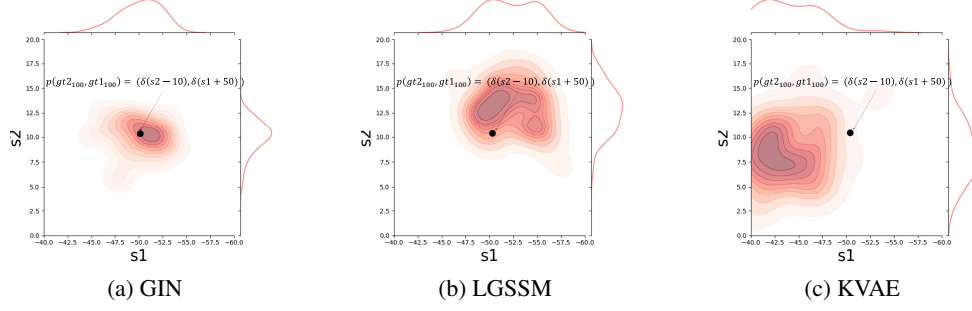

Figure 24: Generated samples from the trained smoothened joint distribution of the visual odometry joint position, $(s1, s2)$, at 100-th time step for the GIN, LGSSM and KVAE, respectively. The ground truth is shown with a black point.

## A.11   Algorithms and python inference code

---

**Algorithm**   Training the GIN

---

**Input:** Observations $\mathbf{o}_{1:T}$, last posteriors $(\hat{\mu}_{0:T-1|0:T-1}, \hat{\mathbf{\Sigma}}_{0:T-1|0:T-1})$

$(\mathbf{x}_{0:T-1|0:T-1}) = Sample((\hat{\mu}_{0:T-1|0:T-1}, \hat{\mathbf{\Sigma}}_{0:T-1|0:T-1}))$

$\alpha_{1:T} = Dynamic\ Net\ (\mathbf{x}_{0:T-1|0:T-1})$

Obtain $\hat{\mathbf{F}}_{1:T}$ and $\hat{\mathbf{H}}_{1:T}$ by (5)

$(\hat{\mu}_{1:T|0:T-1}, \hat{\mathbf{\Sigma}}_{1:T|0:T-1}) = Prediction\ (\hat{\mu}_{0:T-1|0:T-1}, \hat{\mathbf{\Sigma}}_{0:T-1|0:T-1}, \hat{\mathbf{F}}_{1:T})$

$(\mathbf{w}_{1:T}, \mathbf{r}_{1:T}) = encoder\ (\mathbf{o}_{1:T})$

$\hat{\mathbf{K}}_{1:T} = \hat{\mathbf{\Sigma}}_{1:T|0:T-1}\hat{\mathbf{H}}_{1:T}\ \mathbf{M}_{1:T}\mathbf{M}_{1:T}^T, \quad \mathbf{M}_{1:T} = GRU^{KG}(Conv(\hat{\mathbf{\Sigma}}_{1:T|0:T-1}), \mathbf{r}_{1:T}\ )$

$\hat{\mathbf{J}}_{1:T} = \hat{\mathbf{\Sigma}}_{1:T|1:T}\hat{\mathbf{F}}_{1:T}^T\mathbf{N}_{1:T}\mathbf{N}_{1:T}^T, \quad \mathbf{N}_{1:T} = GRU^{SG}\ (Conv(\hat{\mathbf{\Sigma}}_{1:T|0:T-1}))$

$(\hat{\mu}_{1:T|1:T}, \hat{\mathbf{\Sigma}}_{1:T|1:T}) = Filtering\ (\hat{\mu}_{1:T|0:T-1}, \hat{\mathbf{\Sigma}}_{1:T|0:T-1}, \hat{\mathbf{K}}_{1:T}, \mathbf{w}_{1:T}, \hat{\mathbf{H}}_{1:T})$

$(\hat{\mu}_{1:T|T}, \hat{\mathbf{\Sigma}}_{1:T|T}) = Smoothing\ (\hat{\mu}_{1:T|1:T}, \hat{\mathbf{\Sigma}}_{1:T|1:T}, \hat{\mathbf{J}}_{1:T}, \hat{\mathbf{F}}_{1:T})$

**if** (image imputation) **then**

    $\mathbf{o}_{1:T} = decoder\ (\hat{\mu}_{1:T|T}, \hat{\mathbf{\Sigma}}_{1:T|T})$

    $\mathcal{L}(\mathbf{o}_{1:T}) =$ Lower bound likelihood $(\mathbf{o}_{1:T})$ by using (10)

**end if**

**if** (state estimation) **then**

    $\mathbf{s}_{1:T} = decoder\ (\hat{\mu}_{1:T|T}, \hat{\mathbf{\Sigma}}_{1:T|T})$

    $\mathcal{L}(\mathbf{s}_{1:T}|\mathbf{o}_{1:T}) =$ Lower bound likelihood $(\mathbf{s}_{1:T})$ by using (11)

**end if**

Backward Propagation by updating GRU cells with the steps in **Theorem 3** or **Theorem 4**.

---

Table 8: MSE for single pendulum experiment.

| Model | MSE |
|---|---|
| LSTM (units = 25, $m = 15$) | 0.092±0.003 |
| LSTM (units = 25, $m = 20$) | 0.092±0.005 |
| LSTM (units = 25, $m = 40$) | 0.090±0.005 |
| LSTM (units = 100, $m = 15$) | 0.089±0.002 |
| LSTM (units = 100, $m = 20$) | 0.089±0.005 |
| LSTM (units = 100, $m = 40$) | 0.090±0.004 |
| GRU (units = 30, $m = 15$) | 0.095±0.006 |
| GRU (units = 30, $m = 20$) | 0.093±0.002 |
| GRU (units = 30, $m = 40$) | 0.094±0.005 |
| GRU (units = 100, $m = 15$) | 0.091±0.002 |
| GRU (units = 100, $m = 20$) | 0.092±0.004 |
| GRU (units = 100, $m = 40$) | 0.091±0.008 |

| Model | $\hat{\mathbf{F}}(\mathbf{Q})$ | MLP($\mathbf{Q}$) | GRU($\mathbf{Q}$) |
|---|---|---|---|
| LGSSM filter($m = 15$, $n = 30$, $K = 10$) | 0.089±0.009 | 0.088±0.005 | 0.088±0.006 |
| LGSSM filter($m = 15$, $n = 30$, $K = 15$) | 0.088±0.011 | 0.087±0.007 | 0.086±0.004 |
| LGSSM filter($m = 15$, $n = 45$, $K = 10$) | 0.085±0.004 | 0.084±0.007 | 0.084±0.009 |
| LGSSM filter($m = 15$, $n = 45$, $K = 15$) | 0.084±0.005 | 0.083±0.004 | 0.082±0.004 |
| LGSSM filter($m = 20$, $n = 40$, $K = 10$) | 0.084±0.009 | 0.082±0.014 | 0.082±0.011 |
| LGSSM filter($m = 20$, $n = 40$, $K = 15$) | 0.083±0.012 | 0.081±0.005 | 0.080±0.014 |
| LGSSM filter($m = 20$, $n = 60$, $K = 10$) | 0.079±0.005 | 0.078±0.012 | 0.076±0.009 |
| LGSSM filter($m = 20$, $n = 60$, $K = 15$) | 0.077±0.006 | 0.075±0.011 | 0.074±0.008 |
| LGSSM smooth($m = 15$, $n = 30$, $K = 10$) | 0.086±0.011 | 0.083±0.004 | 0.084±0.007 |
| LGSSM smooth($m = 15$, $n = 30$, $K = 15$) | 0.085±0.012 | 0.084±0.008 | 0.083±0.012 |
| LGSSM smooth($m = 15$, $n = 45$, $K = 10$) | 0.081±0.008 | 0.080±0.009 | 0.079±0.003 |
| LGSSM smooth($m = 15$, $n = 45$, $K = 15$) | 0.081±0.014 | 0.078±0.007 | 0.077±0.011 |
| LGSSM smooth($m = 20$, $n = 40$, $K = 10$) | 0.082±0.005 | 0.078±0.004 | 0.076±0.013 |
| LGSSM smooth($m = 20$, $n = 40$, $K = 15$) | 0.080±0.003 | 0.076±0.006 | 0.074±0.010 |
| LGSSM smooth($m = 20$, $n = 60$, $K = 10$) | 0.076±0.008 | 0.073±0.002 | 0.070±0.009 |
| LGSSM smooth($m = 20$, $n = 60$, $K = 15$) | 0.073±0.013 | 0.071±0.011 | 0.068±0.013 |
| GIN filter($m = 15$, $n = 30$, $K = 10$) | 0.078±0.013 | 0.076±0.005 | 0.075±0.004 |
| GIN filter($m = 15$, $n = 30$, $K = 15$) | 0.078±0.014 | 0.075±0.009 | 0.074±0.012 |
| GIN filter($m = 15$, $n = 45$, $K = 10$) | 0.074±0.010 | 0.073±0.008 | 0.072±0.009 |
| GIN filter($m = 15$, $n = 45$, $K = 15$) | 0.073±0.015 | 0.074±0.011 | 0.071±0.005 |
| GIN filter($m = 20$, $n = 40$, $K = 10$) | 0.072±0.005 | 0.072±0.008 | 0.070±0.002 |
| GIN filter($m = 20$, $n = 40$, $K = 15$) | 0.071±0.007 | 0.071±0.004 | 0.071±0.009 |
| GIN filter($m = 20$, $n = 60$, $K = 10$) | 0.067±0.009 | 0.066±0.005 | 0.065±0.006 |
| GIN filter($m = 20$, $n = 60$, $K = 15$) | 0.065±0.013 | 0.064±0.009 | 0.063±0.010 |
| GIN smooth($m = 15$, $n = 30$, $K = 10$) | 0.071±0.007 | 0.070±0.003 | 0.068±0.009 |
| GIN smooth($m = 15$, $n = 30$, $K = 15$) | 0.070±0.008 | 0.068±0.011 | 0.068±0.007 |
| GIN smooth($m = 15$, $n = 45$, $K = 10$) | 0.065±0.011 | 0.065±0.009 | 0.064±0.012 |
| GIN smooth($m = 15$, $n = 45$, $K = 15$) | 0.064±0.008 | 0.066±0.007 | 0.063±0.009 |
| GIN smooth($m = 20$, $n = 40$, $K = 10$) | 0.064±0.005 | 0.065±0.003 | 0.062±0.008 |
| GIN smooth($m = 20$, $n = 40$, $K = 15$) | 0.063±0.004 | 0.064±0.011 | 0.063±0.007 |
| GIN smooth($m = 20$, $n = 60$, $K = 10$) | 0.059±0.009 | 0.061±0.012 | 0.057±0.006 |
| GIN smooth($m = 20$, $n = 60$, $K = 15$) | 0.058±0.005 | 0.057±0.009 | 0.056±0.004 |

Table 9: MSE for double pendulum experiment.

| Model | MSE |
|---|---|
| LSTM (units = 50, $m = 15$) | 0.172±0.012 |
| LSTM (units = 50, $m = 20$) | 0.166±0.009 |
| LSTM (units = 50, $m = 40$) | 0.167±0.011 |
| LSTM (units = 100, $m = 15$) | 0.164±0.006 |
| LSTM (units = 100, $m = 20$) | 0.162±0.009 |
| LSTM (units = 100, $m = 40$) | 0.159±0.010 |
| GRU (units = 50, $m = 15$) | 0.194±0.014 |
| GRU (units = 50, $m = 20$) | 0.189±0.013 |
| GRU (units = 50, $m = 40$) | 0.188±0.015 |
| GRU (units = 100, $m = 15$) | 0.173±0.009 |
| GRU (units = 100, $m = 20$) | 0.169±0.014 |
| GRU (units = 100, $m = 40$) | 0.166±0.018 |

| Model | $\hat{\mathbf{F}}(\mathbf{Q})$ | MLP($\mathbf{Q}$) | GRU($\mathbf{Q}$) |
|---|---|---|---|
| LGSSM filter($m = 15, n = 30, K = 10$) | 0.154±0.013 | 0.159±0.021 | 0.153±0.009 |
| LGSSM filter($m = 15, n = 30, K = 15$) | 0.152±0.008 | 0.153±0.010 | 0.152±0.012 |
| LGSSM filter($m = 15, n = 45, K = 10$) | 0.144±0.011 | 0.141±0.015 | 0.139±0.013 |
| LGSSM filter($m = 15, n = 45, K = 15$) | 0.142±0.007 | 0.138±0.012 | 0.137±0.017 |
| LGSSM filter($m = 20, n = 40, K = 10$) | 0.144±0.012 | 0.137±0.009 | 0.138±0.013 |
| LGSSM filter($m = 20, n = 40, K = 15$) | 0.141±0.007 | 0.137±0.008 | 0.136±0.016 |
| LGSSM filter($m = 20, n = 60, K = 10$) | 0.129±0.009 | 0.126±0.014 | 0.122±0.015 |
| LGSSM filter($m = 20, n = 60, K = 15$) | 0.127±0.012 | 0.124±0.013 | 0.119±0.009 |
| LGSSM smooth($m = 15, n = 30, K = 10$) | 0.147±0.009 | 0.148±0.014 | 0.144±0.015 |
| LGSSM smooth($m = 15, n = 30, K = 15$) | 0.146±0.014 | 0.146±0.013 | 0.142±0.017 |
| LGSSM smooth($m = 15, n = 45, K = 10$) | 0.139±0.017 | 0.136±0.009 | 0.133±0.017 |
| LGSSM smooth($m = 15, n = 45, K = 15$) | 0.137±0.009 | 0.135±0.017 | 0.133±0.012 |
| LGSSM smooth($m = 20, n = 40, K = 10$) | 0.136±0.014 | 0.131±0.022 | 0.132±0.011 |
| LGSSM smooth($m = 20, n = 40, K = 15$) | 0.134±0.011 | 0.129±0.014 | 0.129±0.022 |
| LGSSM smooth($m = 20, n = 60, K = 10$) | 0.123±0.019 | 0.116±0.016 | 0.115±0.013 |
| LGSSM smooth($m = 20, n = 60, K = 15$) | 0.120±0.010 | 0.112±0.009 | 0.108±0.014 |
| GIN filter($m = 15, n = 30, K = 10$) | 0.126±0.014 | 0.125±0.012 | 0.125±0.011 |
| GIN filter($m = 15, n = 30, K = 15$) | 0.124±0.015 | 0.124±0.019 | 0.121±0.009 |
| GIN filter($m = 15, n = 45, K = 10$) | 0.115±0.011 | 0.114±0.015 | 0.110±0.017 |
| GIN filter($m = 15, n = 45, K = 15$) | 0.114±0.019 | 0.112±0.020 | 0.110±0.011 |
| GIN filter($m = 20, n = 40, K = 10$) | 0.113±0.013 | 0.111±0.009 | 0.111±0.013 |
| GIN filter($m = 20, n = 40, K = 15$) | 0.111±0.009 | 0.109±0.018 | 0.108±0.009 |
| GIN filter($m = 20, n = 60, K = 10$) | 0.099±0.018 | 0.094±0.017 | 0.095±0.021 |
| GIN filter($m = 20, n = 60, K = 15$) | 0.097±0.009 | 0.093±0.009 | 0.091±0.008 |
| GIN smooth($m = 15, n = 30, K = 10$) | 0.115±0.011 | 0.116±0.009 | 0.113±0.017 |
| GIN smooth($m = 15, n = 30, K = 15$) | 0.113±0.015 | 0.113±0.018 | 0.112±0.014 |
| GIN smooth($m = 15, n = 45, K = 10$) | 0.105±0.009 | 0.101±0.014 | 0.101±0.015 |
| GIN smooth($m = 15, n = 45, K = 15$) | 0.102±0.013 | 0.100±0.011 | 0.098±0.008 |
| GIN smooth($m = 20, n = 40, K = 10$) | 0.101±0.008 | 0.098±0.010 | 0.094±0.015 |
| GIN smooth($m = 20, n = 40, K = 15$) | 0.098±0.017 | 0.095±0.014 | 0.092±0.007 |
| GIN smooth($m = 20, n = 60, K = 10$) | 0.086±0.013 | 0.081±0.008 | 0.079±0.009 |
| GIN smooth($m = 20, n = 60, K = 15$) | 0.083±0.009 | 0.079±0.006 | 0.076±0.013 |

Table 10: MSE for bouncing ball experiment.

| Model | MSE |
|---|---|
| LSTM (units = 50, $m = 15$) | 3.724±0.195 |
| LSTM (units = 50, $m = 20$) | 3.698±0.127 |
| LSTM (units = 50, $m = 40$) | 3.684±0.152 |
| LSTM (units = 100, $m = 15$) | 3.476±0.191 |
| LSTM (units = 100, $m = 20$) | 3.39±0.282 |
| LSTM (units = 100, $m = 40$) | 3.28±0.249 |
| GRU (units = 50, $m = 15$) | 3.941±0.378 |
| GRU (units = 50, $m = 20$) | 3.725±0.491 |
| GRU (units = 50, $m = 40$) | 3.65±0.454 |
| GRU (units = 100, $m = 15$) | 3.631±0.392 |
| GRU (units = 100, $m = 20$) | 3.512±0.466 |
| GRU (units = 100, $m = 40$) | 3.389±0.286 |

| Model | $\hat{\mathbf{F}}(\mathbf{Q})$ | MLP($\mathbf{Q}$) | GRU($\mathbf{Q}$) |
|---|---|---|---|
| LGSSM filter($m = 15$, $n = 30$, $K = 10$) | 0.559±0.057 | 0.557±0.039 | 0.543±0.049 |
| LGSSM filter($m = 15$, $n = 30$, $K = 15$) | 0.552±0.051 | 0.547±0.019 | 0.531±0.047 |
| LGSSM filter($m = 15$, $n = 45$, $K = 10$) | 0.541±0.044 | 0.543±0.042 | 0.524±0.061 |
| LGSSM filter($m = 15$, $n = 45$, $K = 15$) | 0.539±0.051 | 0.538±0.042 | 0.519±0.039 |
| LGSSM filter($m = 20$, $n = 40$, $K = 10$) | 0.542±0.072 | 0.536±0.029 | 0.518±0.050 |
| LGSSM filter($m = 20$, $n = 40$, $K = 15$) | 0.524±0.049 | 0.537±0.061 | 0.522±0.037 |
| LGSSM filter($m = 20$, $n = 60$, $K = 10$) | 0.517±0.047 | 0.524±0.029 | 0.514±0.060 |
| LGSSM filter($m = 20$, $n = 60$, $K = 15$) | 0.529±0.034 | 0.517±0.039 | 0.512±0.024 |
| LGSSM smooth($m = 15$, $n = 30$, $K = 10$) | 0.485±0.029 | 0.483±0.019 | 0.475±0.028 |
| LGSSM smooth($m = 15$, $n = 30$, $K = 15$) | 0.476±0.036 | 0.479±0.051 | 0.458±0.026 |
| LGSSM smooth($m = 15$, $n = 45$, $K = 10$) | 0.479±0.042 | 0.471±0.045 | 0.468±0.058 |
| LGSSM smooth($m = 15$, $n = 45$, $K = 15$) | 0.462±0.039 | 0.473±0.041 | 0.471±0.024 |
| LGSSM smooth($m = 20$, $n = 40$, $K = 10$) | 0.469±0.021 | 0.475±0.043 | 0.451±0.034 |
| LGSSM smooth($m = 20$, $n = 40$, $K = 15$) | 0.461±0.029 | 0.465±0.037 | 0.455±0.027 |
| LGSSM smooth($m = 20$, $n = 60$, $K = 10$) | 0.460±0.038 | 0.452±0.014 | 0.442±0.058 |
| LGSSM smooth($m = 20$, $n = 60$, $K = 15$) | 0.448±0.027 | 0.452±0.020 | 0.444±0.019 |
| GIN filter($m = 15$, $n = 30$, $K = 10$) | 0.267±0.028 | 0.264±0.027 | 0.258±0.034 |
| GIN filter($m = 15$, $n = 30$, $K = 15$) | 0.265±0.021 | 0.260±0.020 | 0.251±0.014 |
| GIN filter($m = 15$, $n = 45$, $K = 10$) | 0.264±0.024 | 0.259±0.023 | 0.244±0.019 |
| GIN filter($m = 15$, $n = 45$, $K = 15$) | 0.259±0.019 | 0.260±0.029 | 0.245±0.018 |
| GIN filter($m = 20$, $n = 40$, $K = 10$) | 0.255±0.011 | 0.250±0.018 | 0.239±0.051 |
| GIN filter($m = 20$, $n = 40$, $K = 15$) | 0.256±0.029 | 0.243±0.034 | 0.237±0.016 |
| GIN filter($m = 20$, $n = 60$, $K = 10$) | 0.247±0.043 | 0.240±0.018 | 0.231±0.028 |
| GIN filter($m = 20$, $n = 60$, $K = 15$) | 0.244±0.050 | 0.241±0.031 | 0.231±0.037 |
| GIN smooth($m = 15$, $n = 30$, $K = 10$) | 0.148±0.011 | 0.146±0.012 | 0.139±0.014 |
| GIN smooth($m = 15$, $n = 30$, $K = 15$) | 0.145±0.008 | 0.145±0.031 | 0.133±0.017 |
| GIN smooth($m = 15$, $n = 45$, $K = 10$) | 0.140±0.015 | 0.139±0.008 | 0.130±0.028 |
| GIN smooth($m = 15$, $n = 45$, $K = 15$) | 0.133±0.011 | 0.134±0.024 | 0.129±0.015 |
| GIN smooth($m = 20$, $n = 40$, $K = 10$) | 0.121±0.014 | 0.118±0.008 | 0.110±0.016 |
| GIN smooth($m = 20$, $n = 40$, $K = 15$) | 0.115±0.012 | 0.117±0.009 | 0.106±0.014 |
| GIN smooth($m = 20$, $n = 60$, $K = 10$) | 0.115±0.013 | 0.110±0.018 | 0.099±0.008 |
| GIN smooth($m = 20$, $n = 60$, $K = 15$) | 0.112±0.010 | 0.106±0.012 | 0.094±0.011 |

To demonstrate the simplicity of our proposed GIN, we include intuitive inference code with Tensorflow library. The code runs with Python 3.6+. The entire code to reproduce the experiments are available in Github repository.

### A.11.1 Python intuitive code.

```python
# Inference with 🐢 !

import tensorflow.keras as k
import Prediction
import Filtering
import Smoothing
import DynamicsNetwork
import Encoder
import Decoder
import DataGen

class GIN_CELL(k.layers.Layer):
    def __init__(self, initial_states):
        self.mu_tm1_filt, self.Sigma_tm1_filt = initial_states
        self.filter_states = [[self.mu_tm1_filt, self.Sigma_tm1_filt]]
    def call(self, inputs):
        w_1:T, r_1:T = inputs
        for w_t, r_t in (w_1:T, r_1:T):
            x_tm1_filt = sample(self.mu_tm1_filt, self.Sigma_tm1_filt)
            F_hat_t, H_hat_t = DynamicsNetwork(x_tm1_filt)
            mu_t_pri, Sigma_t_pri = Prediction(F_hat_t, H_hat_t,...
                      self.mu_tm1_filt, self.Sigma_tm1_filt)
            mu_t_filt, Sigma_t_filt = Filtering(mu_t_pri,...
                             Sigma_t_pri, w_t, r_t, H_hat_t)
            self.filter_states.append([mu_t_filt, Sigma_t_filt])
            self.mu_tm1_filt = x_t_filt
            self.Sigma_tm1_filt = Sigma_t_filt
        mu_1:T_smooth, Sigma_1:T_smooth = Smoothing( F_hat_1:T,...
                           self.filter_states, Sigma_1:T_pri,)
        return mu_1:T_smooth, Sigma_1:T_smooth

class GIN(k.models.Model):
    def __init__(self, initial_states):
        self.mu_0_filt, self.Sigma_0_filt = initial_states
        self.GIN_CELL_OBJ = self.GIN_CELL(self.x_0_filt, self.
    Sigma_0_filt)
        self.Encoder = Encoder
        self.Decoder = Decoder

    def call(self, o_1:T):
        w_1:T, r_1:T = self.Encoder(o_1:T)
        mu_1:T_smooth, Sigma_1:T_smooth = self.GIN_CELL_OBJ(w_1:T, r_1
    :T)
        o_1:T, s_1:T = self.Decoder(mu_1:T_smooth, Sigma_1:T_smooth)
        return o_1:T, s_1:T

initial_states = mu_0_filt, Sigma_0_filt
GIN_OBJ =GIN(initial_states)

Data = DataGen()
o_1:T, s_1:T = GIN_OBJ(Data)
```
